# Revisiting Differentially Private ReLU Regression

**Meng Ding**
University at Buffalo

**Mingxi Lei**
University at Buffalo

**Liyang Zhu**
KAUST

**Shaowei Wang**
Guangzhou University

**Di Wang**[*]
KAUST

**Jinhui Xu**[*]
University at Buffalo

## Abstract

As one of the most fundamental non-convex learning problems, ReLU regression under differential privacy (DP) constraints, especially in high-dimensional settings, remains a challenging area in privacy-preserving machine learning. Existing results are limited to the assumptions of bounded norm $\|\mathbf{x}\|_2 \leq 1$, which becomes stringent and strong with increasing data dimensionality. In this work, we revisit the problem of DP ReLU regression in overparameterized regimes. We propose two innovative algorithms, DP-GLMtron and DP-TAGLMtron, that outperform the conventional DPSGD. DP-GLMtron is based on a generalized linear model perceptron approach, integrating adaptive clipping and Gaussian mechanism for enhanced privacy. To overcome the constraints of small privacy budgets in DP-GLMtron, represented by $\widetilde{O}(\sqrt{1/N})$ where $N$ is the sample size, we introduce DP-TAGLMtron, which utilizes a tree aggregation protocol to balance privacy and utility effectively, showing that DP-TAGLMtron achieves comparable performance with only an additional factor of $O(\log N)$ in the utility upper bound. Moreover, our theoretical analysis extends beyond Gaussian-like data distributions to settings with eigenvalue decay, showing how data distribution impacts learning in high dimensions. Notably, our findings suggest that the utility bound could be independent of the dimension $d$, even when $d \gg N$. Experiments on synthetic and real-world datasets also validate our results.

## 1 Introduction

Protecting individual privacy in data analysis has emerged as a critical concern. In light of the vast quantities of personal and sensitive information involved, traditional methods of ensuring privacy are encountering significant challenges. Differential Privacy (DP), introduced by [13], has gained widespread recognition as a method for preserving privacy by adding a controlled amount of random noise to the data or query responses, thereby effectively concealing the details of any individual.

Differentially Private Stochastic Optimization (DP-SO) and its empirical form, Differentially Private Empirical Risk Minimization (DP-ERM), are foundational models extensively studied in the DP community. Although there is a substantial body of research on DP-SO and DP-ERM with convex loss functions [10, 46, 7, 16, 35, 37, 6, 9, 26, 36, 38, 39], the understanding of nonconvex optimization in a DP context remains limited. Recent efforts have developed private algorithms with established proven utility bounds for differentially private nonconvex optimization [51, 46, 43, 52, 8, 47]. However, most of the current work employs the gradient norm of the population risk function as a measurement instead of excess population risk, commonly used in convex scenarios. Recently, [33] comprehensively studied different instances of non-convex learning, such as generalized linear

---

[*]Correspondence to: Di Wang <di.wang@kaust.edu.sa>, Jinhui Xu <jinhui@buffalo.edu>

models, ReLU regression, and multi-layer neural networks. However, for each problem, they imposed different strong assumptions, which left a big room to further understanding DP non-convex learning. In this paper, we consider the most fundamental one, the ReLU (Rectified Linear Unit) regression model.

As one of the most fundamental non-convex models, ReLU regression stands out due to its widespread use and effectiveness in deep learning. Despite its prevalence in industry, theoretical exploration of ReLU regression has been relatively limited. [33] studies the DP ReLU regression problem in both well-specified and misspecified settings. However, the problem is still far from well-understood. By and large, several challenges need to be addressed. Their analysis, for instance, heavily relies on strong assumptions about bounded norms of feature vectors and labels, i.e. $\|\mathbf{x}\|_2 \leq O(1)$ and $|y| \leq O(1)$, which does not hold even for normal Gaussian distributions. Secondly, while they provide a utility upper bound of $\widetilde{O}(\frac{1}{\sqrt{N}} + \min\{\frac{d^{1/2}}{(N\varepsilon)^{1/2}}, \frac{1}{(N\varepsilon)^{2/3}}\})$ with sample size $N$ and dimension $d$, their dimension-independent results heavily rely on their data assumption, leading to a dimension-dependent upper bound when considering Bernoulli or $O(1)$-subGaussian data where $\|\mathbf{x}\|_2 \leq O(\sqrt{d})$ with high probability. That means when $d \gg N$, i.e., when in the overparameterization regime, previous results become trivial.

In this paper, we revisit the DP ReLU regression problem in the overparameterized regime. Specifically, we provide a general analysis under the well-specified setting and propose two novel algorithms (DP-GLMtron and DP-TAGLMtron). Our contributions are three-fold, see Table 1 for details.

**1)** We propose an innovative algorithm, DP-GLMtron, built upon the Generalized Linear Model Perceptron (GLM-tron) algorithm of [23]. Specifically, DP-GLMtron operates by making a single pass over the dataset and processes one data point at a time by integrating appropriate clipping and adding Gaussian noise to maintain privacy. Additionally, we provide an excess population risk upper bound of $\widetilde{O}(\frac{D_{\text{eff}}}{N} + \frac{D_{\text{eff}}^{\text{pri}}}{N^2\varepsilon^2})$ where $D_{\text{eff}}$ ($D_{\text{eff}}^{\text{pri}}$) is the (private) effective dimension. It is noticed that $D_{\text{eff}}$ and $D_{\text{eff}}^{\text{pri}}$ are defined in a way that does not directly rely on the dimension $d$. Furthermore, our results are applicable not only to Gaussian-like data, typically considered in the traditional differential privacy community, but also to data distributions that exhibit either polynomial or exponential decay. In these cases, we demonstrate that the utility bound is independent of the dimension $d$, which marks a significant departure from the traditional one (see more discussions in Remark 4). Finally, we also show that our analysis on DP-GLMtron is almost tight by providing a lower bound of $\widetilde{\Omega}(\frac{D_{\text{eff}}}{N} + \frac{D_{\text{eff}}^{\text{pri}}}{N^2\varepsilon^2})$.

**2)** A significant issue with DP-GLMtron is its upper bound only holds with a small privacy budget $\varepsilon = \widetilde{O}(\frac{1}{\sqrt{N}})$ (for further details, see Remark 3), which is due to privacy amplification via shuffling. To tackle this limitation, we propose DP-TAGLMtron, which utilizes the tree aggregation protocol to introduce noise into the sum of mini-batch gradients instead of using privacy amplification. Our analysis reveals that the utility upper bound of DP-TAGLMtron is closely comparable to that of DP-GLMtron only with an additional factor of $O(\log N)$. This finding indicates that DP-TAGLMtron can offer similar performance benefits while potentially being applicable in settings with larger privacy budgets.

**3)** We conducted experiments with both synthetic data and real data on the ReLU regression model across various algorithms and privacy budgets. Specifically, in the first part, synthetic data was generated under two eigenvalue decay scenarios, $\lambda_i \propto i^{-2}$ and $\lambda_i \propto i^{-3}$. The results revealed that faster eigenvalue decay consistently resulted in lower excess risk compared to slower decay, indicating that it may suffer less from dimensionality with respect to utility. Additionally, in both synthetic and real data, DP-GLMtron and DP-TAGLMtron outperformed DP-SGD and DP-FTRL [22], especially as the sample size increased, highlighting the effectiveness of our proposed methods.

Due to space limitations, we have included the algorithms and proofs in the appendix.

## 2 Related Work

**Private Nonconvex Optimization** Previous research focusing on the utility of DP-SO/DP-ERM with convex loss functions predominantly employed excess population risk as the metric of utility. In contrast, for non-convex scenarios, utility assessment can be broadly categorized into three types: first-order stationary-based, second-order stationary-based, and excess population risk. The first-order stationary-based method [43, 53, 34, 8, 52, 50] involves evaluating the $\ell_2$-norm of the gradient of

Table 1: Comparison of our work with related studies on $(\varepsilon,\delta)$-DP ReLU regression in the statistical estimation setting. Here, $N$ represents the sample size, $d$ refers to the dimension and $D_{\text{eff},}, D_{\text{eff}}^{\text{pri}}$ are the (private) effective dimensions. It is noticed that $D_{\text{eff}}$ and $D_{\text{eff}}^{\text{pri}}$ are defined in a way that does not directly rely on the dimension $d$. Instead, they segment the feature space into an effective subspace. For further discussion, please see Remark 4.

| Method | Upper Bound | Lower Bound | Privacy Constraints | Data Assumptions |
|---|---|---|---|---|
| DP-PGD [33] | $\widetilde{O}(\min\{\frac{d^{1/2}}{(N\varepsilon)^{1/2}}, \frac{1}{(N\varepsilon)^{2/3}}\}+\frac{1}{\sqrt{N}})$ | - | - | $\|\mathbf{x}\|_2 \le 1$ |
| **DP-GLMtron** | $\widetilde{O}(\frac{D_{\text{eff}}}{N} + \frac{D_{\text{eff}}^{\text{pri}}}{N^2\varepsilon^2})$ | $\widetilde{\Omega}(\frac{D_{\text{eff}}}{N} + \frac{D_{\text{eff}}^{\text{pri}}}{N^2\varepsilon^2})$ | $\varepsilon = O(\frac{1}{\sqrt{N}})$ | - |
| **DP-TAGLMtron** | $\widetilde{O}(\frac{D_{\text{eff}}}{N} + \frac{D_{\text{eff}}^{\text{pri}}+D_{\text{eff}}}{N^2\varepsilon^2})$ | - | - | - |

the population risk function. This method, however, encounters several challenges. For example, [2] shows that the gradient norm tends to approach zero as the sample size increases indefinitely, while there exists no assurance that a differentially private estimator will converge to or even be near any significant local minimum. The Second-order stationary-based method [42, 45] employs the norm of the gradient and the Hessian matrix minimal eigenvalue of the population risk function. However, this approach is particularly suited to specific scenarios where any second-order stationary point is a local minimum of the problem, and all the local minima are the global minimum, such as matrix completion and dictionary learning. The third approach involves directly employing the excess population risk as the measure of utility [33, 43] and our work aligns with this direction. However, as we mentioned these bounds either need strong assumptions or become trivial when $d \gg N$.

**Private Generalized Linear Models** DP-SO and DP-ERM with generalized linear loss (DP-GLL) and generalized linear models (DP-GLM) have gained significant attention in recent years. The key developments in this area can be regarded as three aspects: 1) For convex loss functions, an early study on DP-GLL [21] revealed that, assuming features have bounded $l_2$ norm, the error bound can reach $\widetilde{O}(\frac{1}{\sqrt{N}\varepsilon})$, contrasting with the general convex DP-ERM bound of $O(\frac{\sqrt{d}}{N\varepsilon})$. A subsequent study [25] found that in constrained cases, the error bound depends on the Gaussian width of the constraint set. [34] improved the bound to $O(\frac{\sqrt{\theta}}{N\varepsilon})$ under the assumption that the data space is a bounded set, where $\theta$ is the rank of the expectation of the data matrix. 2) For constrained DP-GLM, [8] explored various scenarios, including smooth/non-smooth losses and losses in $\ell_p$ space for $1 \le p \le 2$. [5] provided the optimal rates of DP-GLM in unconstrained settings, identifying different rates depending on the nature of the loss function (smooth and non-negative but not necessarily Lipschitz, or Lipschitz). 3) For non-convex losses, [34, 7] provided dimension-independent bounds for the gradient $\ell_2$ norm of the population risk function. As a specific instance of GLM, this work focuses on the overparameterized regime with sub-Gaussian data rather than bounded data.

## 3 Preliminaries

**Notations**: In this paper, we adhere to a consistent notation style for clarity. We use boldface lower letters such as $\mathbf{x}, \mathbf{w}$ for vectors, and boldface capital letters (e.g. $\mathbf{A}, \mathbf{H}$) for matrices. Let $\|\mathbf{A}\|_2$ denote the spectral norm of $\mathbf{A}$. For two matrices $\mathbf{A}$ and $\mathbf{B}$ of appropriate dimension, their inner product is defined as $\langle \mathbf{A}, \mathbf{B}\rangle := \text{tr}(\mathbf{A}^\top \mathbf{B})$. For a positive semi-definite (PSD) matrix $\mathbf{A}$ and a vector $\mathbf{v}$ of appropriate dimension, we write $\|\mathbf{v}\|_{\mathbf{A}}^2 := \mathbf{v}^\top \mathbf{A}\mathbf{v}$. The outer product is denoted by $\otimes$.

Given a dataset $D = \{(\mathbf{x}_i, y_i)\}_{i=1}^N$ with each data point has a feature vector $\mathbf{x}_i \in \mathcal{X} \subseteq \mathbb{R}^d$ and a response variable $y_i \in \mathcal{Y}$. In this paper, we assume that $\{(\mathbf{x}_i, y_i)\}_{i=1}^N$ are i.i.d. sampled from a ReLU regression model, i.e., each $(\mathbf{x}_i, y_i)$ is a realization of the ReLU regression model $y = \text{ReLU}(\mathbf{x}^\top \mathbf{w}_*) + z$, where $\text{ReLU}(\cdot) := \max\{\cdot, 0\}$ is the Rectified Linear Unit (ReLU); $z$ is a zero mean randomized noise; $\mathbf{w}_* \in \mathbb{R}^d$ is the optimal model parameter. Our goal is to output a model $\mathbf{w}_{\text{priv}}$ that maintains privacy while minimizing the excess population risk, i.e., $\mathcal{L}(\mathbf{w}_{\text{priv}}) - \mathcal{L}(\mathbf{w}_*)$, where

$$\mathcal{L}(\mathbf{w}) = \frac{1}{2}\mathbb{E}_{(\mathbf{x},y)\sim\mathcal{D}}[(\text{ReLU}(\mathbf{x}^\top \mathbf{w}) - y)^2]. \tag{1}$$

In this paper, we will focus on classic Differential Privacy to ensure privacy.

**Definition 3.1** (Differential Privacy [13]). Given a data universe $\mathcal{X}$, we say that two datasets $D, D' \subseteq \mathcal{X}$ are neighbors if they differ by only one element, which is denoted as $D \sim D'$. A randomized algorithm $\mathcal{A}$ is $(\varepsilon, \delta)$-differentially private (DP) if for all adjacent datasets $D, D'$ and for all events $S$ in the output space of $\mathcal{A}$, we have $\mathbb{P}(\mathcal{A}(D) \in S) \leq e^{\varepsilon} \cdot \mathbb{P}(\mathcal{A}(D') \in S) + \delta$.

In the following, we will introduce some definitions and assumptions related to the model.

**Definition 3.2** (Well-specified Condition). Assume that there exists a parameter $\mathbf{w}_* \in \mathbb{R}^d$ such that $\mathbb{E}[y \mid \mathbf{x}] = \mathrm{ReLU}(\mathbf{x}^\top \mathbf{w}_*)$, and the variance of the model noise can be denoted by $\sigma^2 := \mathbb{E}\left[(y - \mathrm{ReLU}(\mathbf{x}^\top \mathbf{w}_*))^2\right]$.

In the literature, the well-specified setting is also extensively referred to as the "noisy teacher" setting [18] or the well-structured noise mode [19]. It has been widely studied previously [56, 41, 33, 12].

**Assumption 3.3** (Data Covariance). Define $\mathbf{H} := \mathbb{E}[\mathbf{x}\mathbf{x}^\top]$ as the expected data covariance matrix.

Denote the eigen decomposition of the data covariance by $\mathbf{H} = \sum_i \lambda_i \mathbf{v}_i \mathbf{v}_i^\top$, where $(\lambda_i)_{i \geq 1}$ are eigenvalues in a nonincreasing order and $(\mathbf{v}_i)_{i \geq 1}$ are the corresponding eigenvectors. Define $\mathbf{H}_{k_1:k_2}$ as $\mathbf{H}_{k_1:k_2} := \sum_{k_1 < i \leq k_2} \lambda_i \mathbf{v}_i \mathbf{v}_i^\top$, and allow $k_2 = \infty$ to imply that $\mathbf{H}_{k:\infty} = \sum_{i > k} \lambda_i \mathbf{v}_i \mathbf{v}_i^\top$.

In the case of a Gaussian distribution, the data covariance matrix reduces to an identity matrix scaled by variance.

**Assumption 3.4** (Fourth moment conditions). Define that the fourth moment of $\mathbf{x}$ as $\mathcal{M}$:

(A) There exists a constant $\alpha > 0$ such that for any Positive Semi-Definite (PSD) matrix $\mathbf{A}$, the following holds:
$$\mathbb{E}[\mathbf{x}\mathbf{x}^\top \mathbf{A}\mathbf{x}\mathbf{x}^\top] \preceq \alpha \cdot \mathrm{tr}(\mathbf{H}\mathbf{A}) \cdot \mathbf{H}.$$

(B) There exists a constant $\beta > 0$, such that for every PSD matrix $\mathbf{A}$, the following holds:
$$\mathbb{E}[\mathbf{x}\mathbf{x}^\top \mathbf{A}\mathbf{x}\mathbf{x}^\top] - \mathbf{H}\mathbf{A}\mathbf{H} \succeq \beta \cdot \mathrm{tr}(\mathbf{H}\mathbf{A}) \cdot \mathbf{H}.$$

**Remark 1.** For Gaussian distribution, it can be verified that Assumption 3.4 holds with $\alpha = 3$ and $\beta = 1$ [56]. Furthermore, the assumption of sub-Gaussianity in data, specifically that $\mathbf{H}^{-\frac{1}{2}}\mathbf{x}$ is a sub-Gaussian random vector, as posited in previous work [44, 41, 54, 28, 55, 20, 32, 31] for private linear regression model, can be shown to imply Assumption 3.4 (A) [56]. It is important to note that Assumption 3.4 (B) is specifically utilized in our analysis for establishing the lower bound only.

**Definition 3.5** $((\mathbf{H}, C_2, a, b)$-Tail). Let both $\mathbf{H}$ and $\mathcal{M}$ exist and are finite. A random vector $\mathbf{x}$ satisfies $(\mathbf{H}, C_2, a, b)$-Tail if the the following holds:

- $\exists a > 0$ s.t. with probability $\geq 1 - b$,
$$\|\mathbf{x}\|_2^2 \leq \mathbb{E}[\|\mathbf{x}\|_2^2] \cdot \log^{2a}(1/b), \tag{2}$$

- We have,
$$\max_{\mathbf{v}, \|\mathbf{v}\|=1} \mathbb{E}[\exp((\frac{|\langle \mathbf{x}, \mathbf{v} \rangle|^2}{C_2^2 \|\mathbf{H}\|_2})^{1/2a})] \leq 1,$$
That is, for any fixed $\mathbf{v}$, with probability $\geq 1 - b$:
$$(\langle \mathbf{x}, \mathbf{v} \rangle)^2 \leq C_2^2 \|\mathbf{H}\|_2 \|\mathbf{v}\|^2 \log^{2a}(1/b).$$

Both conditions above provide Gaussian-like tail bounds determining the resilience of the dataset. Note that Definition 3.5 is widely used in the recent literature on DP analysis for the sub-Gaussian type of data such as [41, 29, 27]. In this work, we will assume each sample $\mathbf{x}$ satisfying $(\mathbf{H}, C_2, a, b)$-Tail and the distribution of the inherent noise $z$ satisfies $(\sigma^2, C_2, a, b)$-Tail, which could capture more statistical properties beyond expectations. Additionally, according to Assumption 3.4 (A), it holds that $\|\mathbf{x}\|_2^2 \leq \alpha \, \mathrm{tr}(\mathbf{H}) \cdot \log^{2a}(1/b)$ in Equation (2).

**Assumption 3.6** (Symmetricity conditions). Assume that for every $\mathbf{u}, \mathbf{v} \in \mathbb{R}^d$, it holds that:
$$\mathbb{E}[\mathbf{x}\mathbf{x}^\top \cdot \mathbb{1}[\mathbf{x}^\top \mathbf{u} > 0, \mathbf{x}^\top \mathbf{v} > 0]] = \mathbb{E}[\mathbf{x}\mathbf{x}^\top \cdot \mathbb{1}[\mathbf{x}^\top \mathbf{u} < 0, \mathbf{x}^\top \mathbf{v} < 0]]$$
$$\mathbb{E}[(\mathbf{x}^\top \mathbf{v})^2 \mathbf{x}\mathbf{x}^\top \cdot \mathbb{1}[\mathbf{x}^\top \mathbf{u} > 0, \mathbf{x}^\top \mathbf{v} > 0]] = \mathbb{E}[(\mathbf{x}^\top \mathbf{v})^2 \mathbf{x}\mathbf{x}^\top \cdot \mathbb{1}[\mathbf{x}^\top \mathbf{u} < 0, \mathbf{x}^\top \mathbf{v} < 0]].$$

**Remark 2.** Here we require that both the second and fourth moments of $\mathbf{x}$ remain symmetric. It can be shown that Assumption 3.6 is satisfied when $\mathbf{x}$ and $-\mathbf{x}$ follow the same distribution, which can apply to symmetric sub-Gaussian such as symmetric Bernoulli or Gaussian distributions.

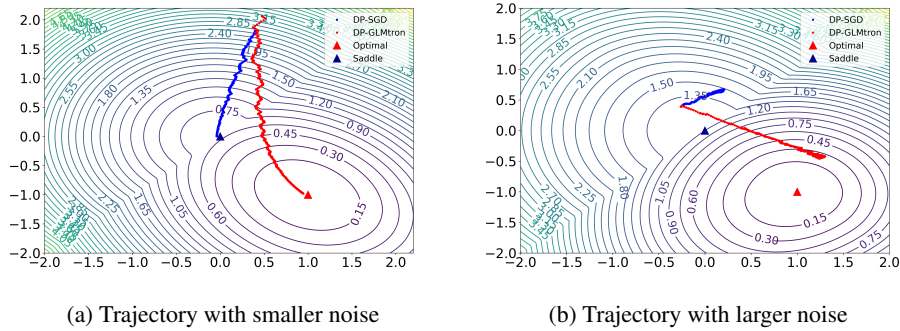

(a) Trajectory with smaller noise        (b) Trajectory with larger noise

Figure 1: Training trajectories of DP-SGD and DP-GLMtron on a 2D noiseless ReLU regression with symmetric Bernoulli data.

## 4 Proposed DP-GLMtron Algorithm

Before presenting the analysis of DP ReLU regression problem, we first introduce our novel algorithm: DP-GLMtron (Algorithm 1), which is inspired by and built upon the Generalized Linear Model Perceptron (GLM-tron) algorithm of [23].

The fundamental distinction between SGD and GLMtron lies in their respective update rules. Specifically, it takes the following rules:

$$\text{SGD:} \quad \mathbf{w}_t = \mathbf{w}_{t-1} - \eta \cdot (\text{ReLU}(\mathbf{x}_t^\top \mathbf{w}_{t-1}) - y_t)\mathbf{x}_t \cdot \mathbb{1}[\mathbf{x}_t^\top \mathbf{w}_{t-1} > 0]$$
$$\text{GLMtron:} \quad \mathbf{w}_t = \mathbf{w}_{t-1} - \eta \cdot (\text{ReLU}(\mathbf{x}_t^\top \mathbf{w}_{t-1}) - y_t)\mathbf{x}_t. \tag{3}$$

where the algorithm commence from an initial point $\mathbf{w}_0$ and iterate from $t = 0$ to $t = N$ with stepsize $\eta$. In contrast to the update rule of SGD, GLMtron diverges by changing the derivative of the ReLU function in its update mechanism. Typically, in neural network training, the gradient of the activation function, such as ReLU, is a critical component of the backpropagation process. The exclusion of this derivative in GLMtron's framework not only simplifies the computational process but also enhances efficiency. Our motivation for developing the DP-GLMtron algorithm stems from observations regarding the limitations of DP-SGD, as it sometimes fails to reach the optimal solution and can struggle with convergence under conditions of high noise, often settling at a saddle point instead (as illustrated in Figure 4). This issue is particularly pronounced when considering a low privacy budget. Furthermore, [23] demonstrates that this specific omission plays a significant role in GLMtron's ability to efficiently identify a predictor that closely approximates the optimal solution.

Building upon these foundations, we now present the detailed implementation of our proposed DP-GLMtron. For Algorithm 1, we begin with a random permutation of the dataset, ensuring that each data point is treated independently and uniformly. Also, such shuffling can amplify privacy[17]. Subsequently, the algorithm performs a single pass over the dataset. Throughout this pass, we need to determine the clipping threshold before undergoing iterative updates for $\mathbf{w}$. Establishing an appropriate clipping threshold is crucial for balancing the trade-off between utility and privacy. If the clipping threshold is set excessively low, it results in loss of gradient information, consequently leading to high bias [30, 3]. Consequently, we propose an advanced Algorithm 2 based on the residual estimator in [28] to achieve an adaptive clipping [4] by utilizing a small batch of data points.

Recall that the clipping term is a product of the residual, $(\text{ReLU}(\mathbf{x}_t^\top \mathbf{w}_t) - \mathbf{y}_t)$, and the covariate, $\mathbf{x}_t^\top$. Based on Definition 3.5 and Assumption 3.4, we understand that $\|\mathbf{x}_t\|_2^2 \leq \alpha \operatorname{tr}(\mathbf{H}) \cdot \log^{2a}(1/b)$ with a probability of at least $1 - b$. Thus, it is sufficient to get an upper bound of $|\text{ReLU}(\mathbf{x}_t^\top \mathbf{w}_t) - \mathbf{y}_t|$. To bound this, we consider three cases, **1)** $\mathbf{x}_t^\top \mathbf{w}_t > 0, \mathbf{x}_t^\top \mathbf{w}_* > 0$; **2)** $\mathbf{x}_t^\top \mathbf{w}_t < 0, \mathbf{x}_t^\top \mathbf{w}_* > 0$; **3)** $\mathbf{x}_t^\top \mathbf{w}_t > 0, \mathbf{x}_t^\top \mathbf{w}_* < 0$, corresponding to $d_t^1, d_t^2$, and $d_t^3$ in step 2 of Algorithm 2. It is noticed that the inherent model noise is not considered when initially calculating the residual for the third case at step 2, while this noise is taken into account at step 15. Moreover, when considering a special case that $\mathbf{x}_t^\top \mathbf{w}_t < 0, \mathbf{x}_t^\top \mathbf{w}_* < 0$, the clipping threshold is integrated into the third scenario. For each residual, we first partition the dataset into $K$ batches, compute an estimate for each batch, and form a histogram with over those $K$ estimates. We then apply a private histogram mechanism with geometrically increasing bin sizes. The bin with the most estimates is used, ensuring a constant factor

approximation of the distance to the optimum. Finally, we take the maximum among the three private residual estimates.

Upon determining the clipping threshold $\gamma_t$, it is subsequently incorporated into the existing clipping list $\gamma$. Following this, Gaussian noise with a standard deviation of $2f\gamma_t$ is introduced to the clipped $l_t$ and then the weight vector $\mathbf{w}_{t+1}$ will be updated. Finally, the algorithm culminates by computing the average of the iterates.

**Theorem 4.1** (Privacy Guarantee). *Suppose DP-GLMtron is applied to $N$ input samples with a noise multiplier set to $f = O(\frac{\log(N/\delta)}{\varepsilon\sqrt{N}})$. Under these conditions, the algorithm satisfies $(\varepsilon, \delta)$-differential privacy for $\varepsilon = O(\sqrt{\frac{\log(N/\delta)}{N}})$, with $\varepsilon' = \frac{\varepsilon}{\sqrt{8N\log(2/\delta)}}$, $\delta' = \frac{\delta}{2N}$ in Algorithm 2.*

**Remark 3.** For general data distributions, DP-GLMtron necessitates a more complex approach to clipping decisions. To this end, we incorporate Algorithm 2, which uses the magnitude of the residual along with privacy parameters $\varepsilon'$ and $\delta'$, as a means to estimate the private threshold. Additionally, we take only one pass over random shuffling data for Algorithm 1 [15, 17, 41], which is a well-known method of privacy amplification. Hence, we need to add noise $O(\frac{\gamma_t}{\varepsilon\sqrt{N}})$, where $\gamma_t$ is the clipping threshold and $\varepsilon$ must be restricted to $1/\sqrt{N}$, to satisfy the requirement of [17] and to ensure differential privacy.

**Theorem 4.2** (Utility Guarantee Upper). *Consider a dataset $D = \{(\mathbf{x}_i, y_i)\}_{i=1}^N$, where each $\mathbf{x}_i$ is drawn from a distribution $\mathcal{D}$ satisfying $(\mathbf{H}, C_2, a, b)$-Tail and the distribution of the inherent noise $z$ satisfies $(\sigma^2, C_2, a, b)$-Tail. Given Assumptions 3.4 and 3.6, if we set a step size $\eta \leq \frac{1}{2\alpha \operatorname{tr}(\mathbf{H})\log^{2a}(N)}$, a noise multiplier $f = O(\frac{\log(N/\delta)}{\varepsilon\sqrt{N}})$, estimating data size $m = \Omega(\frac{\log(N)\log(N/\delta')}{\varepsilon'})$, then with a probability of $1 - 1/N$, the output of Algorithm 1 will satisfy:*

$$\mathbb{E}[L(\overline{\mathbf{w}}_N)] - L(\mathbf{w}_*) \lesssim bias + variance + privacy,$$

*where the bias, variance, and private terms are upper bounded by*

$$bias \lesssim \frac{1}{\eta^2 N^2} \cdot \|\mathbf{w}_0 - \mathbf{w}_*\|_{\mathbf{H}_{0:k^*}^{-1}}^2 + \|\mathbf{w}_0 - \mathbf{w}_*\|_{\mathbf{H}_{k^*:\infty}}^2 + (\frac{\alpha}{N}\|\mathbf{w}_0 - \mathbf{w}_*\|_{\frac{\mathbf{I}_{0:k^*}}{N\eta} + \mathbf{H}_{k^*:\infty}}^2) \cdot D_{eff}$$

$$variance \lesssim \frac{\sigma^2}{N} \cdot D_{eff}, \quad private \lesssim \frac{\Gamma^2 f^2}{N} \cdot D_{eff}^{pri}$$

*where the effective dimensions are given by*

$$D_{eff} = k^* + \frac{N^2\eta^2}{4}\sum_{i>k^*}\lambda_i^2, \quad D_{eff}^{pri} = \sum_{i<k^*}\lambda_i^{-1} + \frac{N^2\eta^2}{4} \cdot \sum_{i>k^*}\lambda_i$$

*with $k^* = \max\{k : \lambda_k \geq \frac{2}{\eta N}\}$ and $\Gamma = \max\{\gamma_t\}_{t=0}^{T-1}$.*

**Remark 4.** The utility bound is typically decomposed into three terms: the bias term, which is associated with the initial value of the model; the variance term, stemming from the inherent noise intrinsic to the model itself; and the privacy term, introduced by the privacy-preserving mechanism. Notably, the results contain two crucial factors: the effective dimension $D_{\text{eff}}$ and the private effective dimension $D_{\text{eff}}^{\text{pri}}$, which can be regarded as the function of data covariance matrix's eigenvalues.

**Technical Understanding Under Simplified Cases** To provide a more intuitive explanation, we explore two simplified scenarios. **1)** $\|\mathbf{x}\|_2 \leq 1$: In this scenario, where the norm of the data is constrained, the trace of the covariance matrix $\operatorname{tr}(\mathbf{H}) \leq 1$. This condition indicates that the sum of the eigenvalues, $\lambda_i$, is less than 1. Given that the eigenvalues are ordered non-increasingly (refer to Assumption 3.3), the majority of these eigenvalues must be very small. Assuming the step size $\eta$ is sufficiently small and scaled with the square root of the data size, $\sqrt{N}$, the optimal number of iterations $k^*$ must be confined within a scope proportional to $\sqrt{N}$. If not, the inequality $\operatorname{tr}(\mathbf{H}) \geq k^* \cdot \frac{2}{N\eta} \geq 1$ would contradict the data assumptions. Furthermore, the clipping threshold, which depends on $\operatorname{tr}(\mathbf{H})$, remains a constant term, and the private effective dimension $D_{\text{eff}}^{\text{pri}}$ is bounded above by $N$. Therefore, the total utility is $\widetilde{O}(\frac{1}{\sqrt{N}} + \frac{\operatorname{tr}(\mathbf{H}^{-1})}{N^2\varepsilon^2})$. **2)** sub-Gaussian tail: In this case, the eigenvalue of the covariance matrix corresponds to the variance of sub-Gaussian data. Both the effective dimension, $D_{\text{eff}}$, and the private effective dimension, $D_{\text{eff}}^{\text{pri}}$, effectively reduce to the feature dimension $d$. Moreover, the clipping threshold contributes a factor of $d$, leading to a total utility bound of $\widetilde{O}(\frac{d}{\sqrt{N}} + \frac{d^2}{N^2\varepsilon^2})$.

**The Role of Eigenvalue in High-dimensional Setting** Existing work in the DP community primarily relies on data assumptions satisfying Gaussian-like distributions, as discussed in Remark 1, which reduces the data covariance matrix to an identity matrix scaled by variance. Such an assumption may cause the case to be overlooked when the spectrum exhibits decay. Theorem 4.2 offers a novel perspective by elucidating the interplay between the feature space and the utility bound. Notice the crucial cut-off index $k^* = \max\{k : \lambda_k \geq \frac{2}{\eta N}\}$, which separates the entire space into a $k^*$ dimensional subspace. It enables us to obtain a bound that does not directly depend on the feature dimension $d$ in the overparameterized regime. To achieve a diminishing bound, it is necessary for $k^*$ to be $o(N)$ and for the tail summation of eigenvalues to be $o(1/N)$, which indicates that it is possible to achieve improved results without suffering the dimension $d$ in the overparameterized regime.

To better understand the effective dimension, we offer examples of data spectra where the excess risk diminishes under certain conditions.

**Corollary 4.3.** *Given the same assumptions in Theorem 4.2 and suppose $\|\mathbf{w}_0 - \mathbf{w}^*\|_2$ is bounded, it holds that:*

1. *If the spectrum of $\mathbf{H}$ satisfies $\lambda_k = k^{-r}$ for some $r > 1$, then with probability at least $1 - 1/N$, the utility is upper bounded by $\widetilde{O}(N^{-\frac{r}{1+r}}(1 + (\varepsilon^{-2}N^{\frac{r}{1+r}})^{\frac{r}{1+2r}}))$.*

2. *If the spectrum of $\mathbf{H}$ satisfies $\lambda_k = e^{-k}$, then with probability at least $1 - 1/N$, the utility is upper bounded by $\widetilde{O}(N^{-1}(1 + (\varepsilon^{-2}N)^{\frac{1}{2}}))$.*

**Remark 5.** Corollary 4.3 indicates that utility bound in the overparameterized setting can still vanish if the spectrum exhibits either polynomial or exponential decay. Specifically, a constant privacy budget $\varepsilon = O(1)$ can yield utility bounds of $\widetilde{\mathcal{O}}(N^{-r/(1+2r)})$ for polynomial decay and $\widetilde{\mathcal{O}}(N^{-1/2})$ for exponential decay. Notably, faster eigenvalue decay (e.g. $r = 2, 3$) consistently resulted in lower excess risk compared to slower decay, indicating that it may suffer less from dimensionality with respect to utility. This insight cannot be derived when considering only traditional Gaussian-like data. Our experiment in Section 6 also validates the conclusion. Moreover, increasing the privacy budget to $\varepsilon = O(N^{r/(2+2r)})$ for polynomial decay and $\varepsilon = O(N^{1/2})$ for exponential decay improves utility bounds to $\widetilde{\mathcal{O}}(N^{-r/(1+r)})$ and $\widetilde{\mathcal{O}}(N^{-1})$, respectively, matching non-private and linear regression scenario results [49, 48].

In the following, we show that our previous analysis for Algorithm 1 is almost tight.

**Theorem 4.4** (Utility Guarantee Lower). *Consider a dataset $D = \{(\mathbf{x}_i, y_i)\}_{i=1}^N$, where each $\mathbf{x}_i$ is drawn from a distribution $\mathcal{D}$ satisfying $(\mathbf{H}, C_2, a, b)$-Tail and the distribution of the inherent noise $z$ satisfies $(\sigma^2, C_2, a, b)$-Tail. Given Assumptions 3.4 and 3.6, if we set the same parameters as in Theorem 4.2, then with a probability of $1 - 1/N$, the output of Algorithm 1 will satisfy:*

$$\mathbb{E}[L(\overline{\mathbf{w}}_N)] - L(\mathbf{w}_*) \gtrsim bias + variance + privacy,$$

*where*

$$bias \gtrsim \frac{1}{\eta^2 N^2} \cdot \|\mathbf{w}_0 - \mathbf{w}_*\|^2_{\mathbf{H}_{0:k^*}^{-1}} + \|\mathbf{w}_0 - \mathbf{w}_*\|^2_{\mathbf{H}_{k^*:\infty}} + (\frac{\beta}{N} \|\mathbf{w}_0 - \mathbf{w}_*\|^2_{\frac{\mathbf{I}_{0:k^*}}{N\eta} + \mathbf{H}_{k^*:\infty}}) \cdot D_{eff}$$

$$variance \gtrsim \frac{\sigma^2}{N} \cdot D_{eff}, \quad private \gtrsim \frac{\widetilde{\Gamma}^2 f^2}{N} \cdot D_{eff}^{pri}$$

*where the effective dimensions, $k_m^*$ are defined same as in Theorem 4.2 and denotes $\widetilde{\Gamma} = \min\{\gamma_t\}_{t=0}^{T-1}$.*

**Remark 6.** Similar to the Theorem 4.2, the lower bound stated here consists of three terms: the bias term, the variance term, and the privacy term. It is noticed that Theorem 4.4 provides the algorithmic lower bound Algorithm 1, applicable in both under-parameterized and over-parameterized settings. Notably, our lower bound aligns with the upper bound, differing only by absolute constants.

## 5 Advanced DP-TAGLMtron

A pivotal concern with Algorithm 1 DP-GLMtron lies in its primary applicability in the setting where the privacy budget $\varepsilon$ is small, which could potentially limit its practical utility (see Remark 3 for more details). To address this challenge, we introduce DP-TAGLMtron (Algorithm 3) in this section, which is designed to attain an improved trade-off between privacy and utility.

In Algorithm 3, the process starts with setting the initial weights. For each iteration $t$, a private residual is determined by Algorithm 2 on the estimating samples, current weights, and privacy parameters. Subsequently, the clipping bound is computed as the product of the private residual and the covariate, analogous to the previous case. Once this clipping bound $\gamma_t$ for the current iteration is computed, it is added to the existing clipping list $\boldsymbol{\gamma}$. With $\gamma_t$ in place, the clipped "gradient" $(\boldsymbol{l}_t)$ is calculated. The maximum value from this list is selected as the clipping threshold $\Gamma$ for our next step. The "gradient" $\boldsymbol{l}_t$ is then forwarded to the differentially private tree-aggregation mechanism with noise multiplier $f$ and the maximal clipping threshold $\Gamma$.

It is noticed that our proposed algorithm, DP-TAGLMtron, does not depend on privacy amplification. Instead, it primarily involves employing tree aggregation [14, 11, 22, 40] to introduce noise into the sum of mini-batch gradients. At the beginning of training, we construct a binary tree consisting of $(2^{\lceil \log_2 N \rceil + 1} - 1)$ nodes, corresponding to $N$ leaves. Each leaf node is assigned a label $\boldsymbol{l}_i$, and the value of each internal node represents the sum of its child nodes. In each iteration, the tree receives a vector $\boldsymbol{l}_t$, derived from the current data pair $(\mathbf{x}_t, y_t)$, which is used to update the tree, incorporating the new node. The final output, denoted as $\boldsymbol{l}'_{\leq t}$, is obtained by aggregating the gradients accumulated in the binary tree and adding Gaussian noise to this aggregate. As a result, we can understand that $\boldsymbol{l}'_{\leq t}$ is the private estimation of $\sum_{i=0}^{t} \boldsymbol{l}_i$ generated by tree aggregation protocol as follows:

$$\boldsymbol{l}'_{\leq t} = \sum_{i=0}^{t} \boldsymbol{l}_i + \mathbf{v}_t, \text{where} \quad \boldsymbol{l}_i = \text{clip}_{\gamma_t}(\mathbf{x}_i(\text{ReLU}(\mathbf{x}_i^\top \mathbf{w}_i) - y_i)),$$

and $\mathbf{v}_t$ is a combination of at most $O(\lceil \log_2 t \rceil)$ Gaussian. Therefore, each $\boldsymbol{l}_i$ influences only its ancestor nodes, affecting at most $\lceil \log t \rceil + 1$ nodes. By introducing Gaussian noise with a standard deviation of $O(\log t / \varepsilon)$ to each node, the entire tree achieves $(\varepsilon, \delta)$-differential privacy.

When receiving the output of Algorithm 4, the optimization objective at step 9 is to determine the weight parameter $\mathbf{w}$. This objective function is composed of two key components: the first term, $(\boldsymbol{l}'_{\leq t}, \mathbf{w})$, represents the inner product of $\boldsymbol{l}'_{\leq t}$ and the weight parameter $\mathbf{w}$. The second term, $\frac{1}{2\eta}\|\mathbf{w}\|_2^2$, introduces a regularization mechanism, where $\eta$ controls the balance between fitting the training data and constraining the magnitude of the parameters to prevent overfitting. One common method to find the optimal solution for this optimization problem is to take the derivative of the objective function and set it to zero. Therefore, for the given problem at step 9, we have the following update:

$$\mathbf{w}_{t+1} \leftarrow -\eta \cdot \boldsymbol{l}'_{\leq t} = -\eta \cdot (\sum_{i=0}^{t} \boldsymbol{l}_i + \mathbf{v}_t).$$

It should be noted that in Algorithm 1, if the initialization is set to $\mathbf{w}_0 = \mathbf{0}$, then the update rule of $\mathbf{w}$ can be considered equivalent to the one presented here, provided that the variance of the noise and the clipping threshold are disregarded. Upon completing all iterations, the final model weights are determined by averaging all the updated weights from each iteration.

**Theorem 5.1** (Privacy Guarantee). *Suppose DP-GLMtron is applied to $N$ input samples with a noise multiplier set to $f = \frac{\sqrt{2\lceil \lg(N+1) \rceil \ln(1/\delta)}}{\varepsilon}$. Under these conditions, the Algorithm 3 satisfies $(\varepsilon, \delta)$-differential privacy, with $\varepsilon' = \frac{\varepsilon}{\sqrt{8N \log(2/\delta)}}$ and $\delta' = \frac{\delta}{2N}$ in Algorithm 2.*

**Remark 7.** Differing from Algorithm 1, the update rule of DP-TAGLMtron mainly utilizes the DP-Tree-Aggregation mechanism, which involves aggregating the "gradients" collected in a binary tree and then adding Gaussian noise to this aggregate. Based on [22], Algorithm 4 achieves $(\varepsilon, \delta)$-differential privacy with appropriate noise multiplier $f = \frac{\sqrt{2\lceil \lg(N+1) \rceil \ln(1/\delta)}}{\varepsilon}$, ensuring that Algorithm 3 also upholds $(\varepsilon, \delta)$-DP under similar settings with $\varepsilon', \delta'$ as Algorithm 2.

**Theorem 5.2** (Utility Guarantee). *Consider a dataset $D = \{(\mathbf{x}_i, y_i)\}_{i=1}^{N}$, where each $\mathbf{x}_i$ is drawn from a distribution $\mathcal{D}$ satisfying $(\mathbf{H}, C_2, a, b)$-Tail and the distribution of the inherent noise $z$ satisfies $(\sigma^2, C_2, a, b)$-Tail. Given Assumptions 3.4 and 3.6 hold, if we set a stepsize $\eta < 1/(\alpha \operatorname{tr}(\mathbf{H}))$, a noise multiplier $f = \frac{\sqrt{2\lceil \lg(N+1) \rceil \ln(1/\delta)}}{\varepsilon}$, then with a probability of $1 - 1/N$, the output of Algorithm 3 will satisfy:*

$$\mathbb{E}[L(\overline{\mathbf{w}}_N)] - L(\mathbf{w}_*) \lesssim bias + variance + privacy,$$

*where*

$$bias \lesssim \frac{1}{\eta^2 N^2} \cdot \|\mathbf{w}_0 - \mathbf{w}_*\|^2_{\mathbf{H}^{-1}_{0:k^*}} + \|\mathbf{w}_0 - \mathbf{w}_*\|^2_{\mathbf{H}_{k^*:\infty}} + (\frac{\alpha}{N}\|\mathbf{w}_0 - \mathbf{w}_*\|^2_{\frac{\mathbf{I}_{0:k^*}}{N\eta} + \mathbf{H}_{k^*:\infty}}) \cdot D_{\textit{eff}}$$

$$variance \lesssim \frac{\sigma^2}{N} \cdot D_{\textit{eff}}, \quad private \lesssim \frac{f^2 \log_2 N}{N} \cdot (\alpha\eta \log_2 N \operatorname{tr}(\mathbf{H}) + \Gamma^2) \cdot (D_{\textit{eff}} + D^{priv}_{\textit{eff}})$$

*where the effective dimensions and $k^*$ are defined same as in Theorem 4.2.*

**Remark 8.** It is noteworthy that in Theorem 5.2, both the bias term and the variance term are consistent with those in Theorem 4.2. This alignment extends to the non-private results, with the only differences being in the absolute constants. Additionally, the private term in Theorem 5.2 closely aligns with that in Theorem 4.2, with the primary difference being a factor of $\log_2 N$. It is intuitive that the tree aggregation mechanism utilized in DP-TAGLMtron indicates that the noise added for privacy is derived from a combination of at most $\lceil \log_2 N + 1 \rceil$ and at least one random Gaussian vector. This aspect suggests that DP-TAGLMtron diverges from the upper bound of DP-GLMtron at most by a factor of $\log_2 N$ and adheres to the same lower bound as DP-GLMtron with a different noise variance. Moreover, in contrast to Algorithm 1, DP-TAGLMtron does not necessitate $\varepsilon = O(1/\sqrt{N})$ to maintain differential privacy, allowing a broader range of privacy budget settings in terms of flexibility and applicability.

## 6    Empirical Simulation

In this section, we first conduct experiments with synthetic data to validate the theoretical insights related to the role of eigenvalues in a high-dimensional setting. Additionally, due to space constraints, we present a real-data experiment on the MNIST dataset in Appendix B to demonstrate the performance of our proposed method.

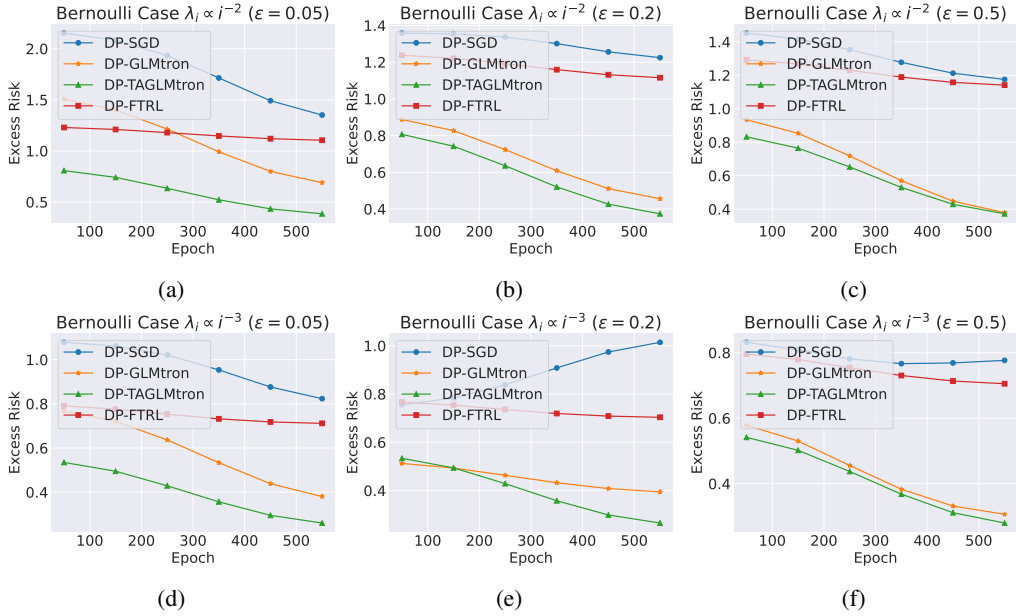

Figure 2: Comparative Performance of Various Differential Privacy Algorithms Across Different Data Spectrum and Privacy Budgets. This figure illustrates the performance of four differential privacy algorithms—DP-SGD, DP-GLMtron, DP-TAGLMtron, and DP-FTRL—on generated Bernoulli distributed data, comparing excess risk across different sample sizes and privacy budgets.

**Experiment Setup.** The experiments were designed with varying privacy budgets ($\varepsilon$) set at 0.05, 0.2, and 0.5 with $\delta = \frac{1}{n^{1.1}}$ to observe the impact of differential privacy constraints on the learning process. We generated the data with two eigenvalue decay scenarios, $\lambda_i \propto i^{-2}, i^{-3}$, simulating different distributions of feature importance. The dimensionality of the data was fixed at 1,024 to maintain consistency across trials. The learning rate was initially set to $10^{-2}$, with $N$ representing

the sample size, which varied from 50 to 550 and increased in steps to 100. We utilized a ReLU regression model with noise to be trained using the previously mentioned algorithms, integrating privacy-preserving noise adjustments. The algorithms underwent a single iteration over generated data. The primary metric for evaluation was the average excess risk across varying dataset sizes and $\varepsilon$ values. For each experiment, we also consider DP-SGD [1] and DP-FTRL [22] as baselines.

**Empirical Findings.** Compared with Figures (2a-2c), which illustrate data with eigenvalue decay $\lambda_i \propto i^{-2}$, Figures (2f-2e), which illustrate data with eigenvalue decay $\lambda_i \propto i^{-3}$ exhibit the lower excess risk under the same training algorithms and conditions regardless of privacy budget. This indicates that fast eigenvalue decay suffers less from the dimensionality with respect to utility, thereby validating our results in Section 4. Additionally, results in both data with eigenvalue decay $\lambda_i \propto i^{-2}$ and data with eigenvalue decay $\lambda_i \propto i^{-3}$ show that DP-GLMtron and DP-TAGLMtron consistently outperform DP-SGD and DP-FTRL regarding excess risk, particularly as the sample size increased. Moreover, it is noticed that the DP-SGD and DP-FTRL exhibit poor convergence performance even with comparatively large data sizes. Finally, in both figures, increasing the privacy budget from $\varepsilon = 0.05$ to $\varepsilon = 0.5$ resulted in lower excess risks for both algorithms. This observation confirms the anticipated trade-off between privacy and learning performance, with a looser privacy constraint leading to more accurate models.

# 7 Conclusions

In conclusion, our study significantly advances the field of differential privacy in ReLU regression problems. Our first proposed algorithm, DP-GLMtron, provides an excess population risk upper bound of $\widetilde{O}(\frac{D_{\text{eff}}}{N} + \frac{D_{\text{eff}}^{\text{pri}}}{N^2 \varepsilon^2})$, offering a novel perspective on utility decomposition and highlighting the impact of eigenvalues in overparameterized settings. Our results apply not only to Gaussian-like data, typically considered in traditional differential privacy studies, but also to data distributions with polynomial or exponential decay. To address limitations with small privacy budgets, we developed DP-TAGLMtron, which offers performance comparable to DP-GLMtron but supports broader privacy budgets. Empirical results from both synthetic and real data experiments demonstrate that DPGLMtron and DP-TAGLMtron consistently outperform DP-SGD and DP-FTRL.

## Acknowledgments

The research of Meng Ding and Jinhui Xu was supported in part by KAUST through grant CRG10-4663.2. Di Wang was supported in part by the baseline funding BAS/1/1689-01-01, funding from the CRG grand URF/1/4663-01-01, REI/1/5232-01-01, REI/1/5332-01-01, FCC/1/1976-49-01 from CBRC of King Abdullah University of Science and Technology (KAUST). Di Wang was also supported by the funding RGC/3/4816-09-01 of the SDAIA-KAUST Center of Excellence in Data Science and Artificial Intelligence (SDAIA-KAUST AI).

## Impact Statements

This paper presents work whose goal is to advance the field of Machine Learning. There are many potential societal consequences of our work, none of which we feel must be specifically highlighted here.

## Limitations

While this study advances previous work by relaxing certain assumptions and providing both synthetic and real-data validations of the theoretical findings, the analysis is specifically confined to the ReLU regression problem. Consequently, it remains uncertain whether these conclusions can be generalized to other nonconvex optimization scenarios. Future research could explore whether similar patterns hold across a broader range of nonconvex problems.

# References

[1] Martin Abadi, Andy Chu, Ian Goodfellow, H Brendan McMahan, Ilya Mironov, Kunal Talwar, and Li Zhang. Deep learning with differential privacy. In *Proceedings of the 2016 ACM SIGSAC conference on computer and communications security*, pages 308–318, 2016.

[2] Naman Agarwal, Zeyuan Allen-Zhu, Brian Bullins, Elad Hazan, and Tengyu Ma. Finding approximate local minima faster than gradient descent. In *Proceedings of the 49th Annual ACM SIGACT Symposium on Theory of Computing*, pages 1195–1199, 2017.

[3] Kareem Amin, Alex Kulesza, Andres Munoz, and Sergei Vassilvtiskii. Bounding user contributions: A bias-variance trade-off in differential privacy. In *International Conference on Machine Learning*, pages 263–271. PMLR, 2019.

[4] Galen Andrew, Om Thakkar, Brendan McMahan, and Swaroop Ramaswamy. Differentially private learning with adaptive clipping. *Advances in Neural Information Processing Systems*, 34:17455–17466, 2021.

[5] Raman Arora, Raef Bassily, Cristóbal Guzmán, Michael Menart, and Enayat Ullah. Differentially private generalized linear models revisited. *Advances in Neural Information Processing Systems*, 35:22505–22517, 2022.

[6] Hilal Asi, John Duchi, Alireza Fallah, Omid Javidbakht, and Kunal Talwar. Private adaptive gradient methods for convex optimization. In *International Conference on Machine Learning*, pages 383–392. PMLR, 2021.

[7] Raef Bassily, Vitaly Feldman, Kunal Talwar, and Abhradeep Guha Thakurta. Private stochastic convex optimization with optimal rates. *Advances in neural information processing systems*, 32, 2019.

[8] Raef Bassily, Cristóbal Guzmán, and Michael Menart. Differentially private stochastic optimization: New results in convex and non-convex settings. *Advances in Neural Information Processing Systems*, 34:9317–9329, 2021.

[9] Raef Bassily, Cristóbal Guzmán, and Anupama Nandi. Non-euclidean differentially private stochastic convex optimization. In *Conference on Learning Theory*, pages 474–499. PMLR, 2021.

[10] Raef Bassily, Adam Smith, and Abhradeep Thakurta. Private empirical risk minimization: Efficient algorithms and tight error bounds. In *2014 IEEE 55th annual symposium on foundations of computer science*, pages 464–473. IEEE, 2014.

[11] T-H Hubert Chan, Elaine Shi, and Dawn Song. Private and continual release of statistics. *ACM Transactions on Information and System Security (TISSEC)*, 14(3):1–24, 2011.

[12] Meng Ding, Kaiyi Ji, Di Wang, and Jinhui Xu. Understanding forgetting in continual learning with linear regression. *arXiv preprint arXiv:2405.17583*, 2024.

[13] Cynthia Dwork, Frank McSherry, Kobbi Nissim, and Adam Smith. Calibrating noise to sensitivity in private data analysis. In *Theory of Cryptography: Third Theory of Cryptography Conference, TCC 2006, New York, NY, USA, March 4-7, 2006. Proceedings 3*, pages 265–284. Springer, 2006.

[14] Cynthia Dwork, Moni Naor, Toniann Pitassi, and Guy N Rothblum. Differential privacy under continual observation. In *Proceedings of the forty-second ACM symposium on Theory of computing*, pages 715–724, 2010.

[15] Úlfar Erlingsson, Ilya Mironov, Ananth Raghunathan, and Shuang Song. That which we call private. *arXiv preprint arXiv:1908.03566*, 2019.

[16] Vitaly Feldman, Tomer Koren, and Kunal Talwar. Private stochastic convex optimization: optimal rates in linear time. In *Proceedings of the 52nd Annual ACM SIGACT Symposium on Theory of Computing*, pages 439–449, 2020.

[17] Vitaly Feldman, Audra McMillan, and Kunal Talwar. Hiding among the clones: A simple and nearly optimal analysis of privacy amplification by shuffling. In *2021 IEEE 62nd Annual Symposium on Foundations of Computer Science (FOCS)*, pages 954–964. IEEE, 2022.

[18] Spencer Frei, Yuan Cao, and Quanquan Gu. Agnostic learning of a single neuron with gradient descent. *Advances in Neural Information Processing Systems*, 33:5417–5428, 2020.

[19] Surbhi Goel and Adam R Klivans. Learning neural networks with two nonlinear layers in polynomial time. In *Conference on Learning Theory*, pages 1470–1499. PMLR, 2019.

[20] Lijie Hu, Shuo Ni, Hanshen Xiao, and Di Wang. High dimensional differentially private stochastic optimization with heavy-tailed data. In *Proceedings of the 41st ACM SIGMOD-SIGACT-SIGAI Symposium on Principles of Database Systems*, pages 227–236, 2022.

[21] Prateek Jain and Abhradeep Guha Thakurta. (near) dimension independent risk bounds for differentially private learning. In *International Conference on Machine Learning*, pages 476–484. PMLR, 2014.

[22] Peter Kairouz, Brendan McMahan, Shuang Song, Om Thakkar, Abhradeep Thakurta, and Zheng Xu. Practical and private (deep) learning without sampling or shuffling. In *International Conference on Machine Learning*, pages 5213–5225. PMLR, 2021.

[23] Sham M Kakade, Varun Kanade, Ohad Shamir, and Adam Kalai. Efficient learning of generalized linear and single index models with isotonic regression. *Advances in Neural Information Processing Systems*, 24, 2011.

[24] Vishesh Karwa and Salil Vadhan. Finite sample differentially private confidence intervals. *arXiv preprint arXiv:1711.03908*, 2017.

[25] Shiva Prasad Kasiviswanathan and Hongxia Jin. Efficient private empirical risk minimization for high-dimensional learning. In *International Conference on Machine Learning*, pages 488–497. PMLR, 2016.

[26] Janardhan Kulkarni, Yin Tat Lee, and Daogao Liu. Private non-smooth empirical risk minimization and stochastic convex optimization in subquadratic steps. *arXiv preprint arXiv:2103.15352*, 2021.

[27] Xiyang Liu, Prateek Jain, Weihao Kong, Sewoong Oh, and Arun Suggala. Label robust and differentially private linear regression: Computational and statistical efficiency. In *Thirty-seventh Conference on Neural Information Processing Systems*, 2023.

[28] Xiyang Liu, Prateek Jain, Weihao Kong, Sewoong Oh, and Arun Sai Suggala. Near optimal private and robust linear regression. *arXiv preprint arXiv:2301.13273*, 2023.

[29] Xiyang Liu, Weihao Kong, Prateek Jain, and Sewoong Oh. Dp-pca: Statistically optimal and differentially private pca. *Advances in Neural Information Processing Systems*, 35:29929–29943, 2022.

[30] H Brendan McMahan, Daniel Ramage, Kunal Talwar, and Li Zhang. Learning differentially private recurrent language models. *arXiv preprint arXiv:1710.06963*, 2017.

[31] Yuan Qiu, Jinyan Liu, and Di Wang. Truthful generalized linear models. In Kristoffer Arnsfelt Hansen, Tracy Xiao Liu, and Azarakhsh Malekian, editors, *Web and Internet Economics - 18th International Conference, WINE 2022, Troy, NY, USA, December 12-15, 2022, Proceedings*, volume 13778 of *Lecture Notes in Computer Science*, pages 369–370. Springer, 2022.

[32] Yuan Qiu, Jinyan Liu, and Di Wang. Truthful and privacy-preserving generalized linear models. *Information and Computation*, 301:105225, 2024.

[33] Hanpu Shen, Cheng-Long Wang, Zihang Xiang, Yiming Ying, and Di Wang. Differentially private non-convex learning for multi-layer neural networks. *arXiv preprint arXiv:2310.08425*, 2023.

[34] Shuang Song, Thomas Steinke, Om Thakkar, and Abhradeep Thakurta. Evading the curse of dimensionality in unconstrained private glms. In *International Conference on Artificial Intelligence and Statistics*, pages 2638–2646. PMLR, 2021.

[35] Shuang Song, Om Thakkar, and Abhradeep Thakurta. Characterizing private clipped gradient descent on convex generalized linear problems. *arXiv preprint arXiv:2006.06783*, 2020.

[36] Jinyan Su, Lijie Hu, and Di Wang. Faster rates of differentially private stochastic convex optimization. *Journal of Machine Learning Research*, 25(114):1–41, 2024.

[37] Jinyan Su and Di Wang. Faster rates of differentially private stochastic convex optimization. *arXiv preprint arXiv*, 2108, 2021.

[38] Jinyan Su, Changhong Zhao, and Di Wang. Differentially private stochastic convex optimization in (non)-euclidean space revisited. In *Uncertainty in Artificial Intelligence*, pages 2026–2035. PMLR, 2023.

[39] Youming Tao, Yulian Wu, Xiuzhen Cheng, and Di Wang. Private stochastic convex optimization and sparse learning with heavy-tailed data revisited. In *IJCAI*, pages 3947–3953. ijcai.org, 2022.

[40] Hoang Tran and Ashok Cutkosky. Momentum aggregation for private non-convex erm. *Advances in Neural Information Processing Systems*, 35:10996–11008, 2022.

[41] Prateek Varshney, Abhradeep Thakurta, and Prateek Jain. (nearly) optimal private linear regression via adaptive clipping. *arXiv preprint arXiv:2207.04686*, 2022.

[42] Di Wang, Changyou Chen, and Jinhui Xu. Differentially private empirical risk minimization with non-convex loss functions. In *International Conference on Machine Learning*, pages 6526–6535. PMLR, 2019.

[43] Di Wang and Jinhui Xu. Differentially private empirical risk minimization with smooth non-convex loss functions: A non-stationary view. In *Proceedings of the AAAI Conference on Artificial Intelligence*, volume 33, pages 1182–1189, 2019.

[44] Di Wang and Jinhui Xu. On sparse linear regression in the local differential privacy model. In *International Conference on Machine Learning*, pages 6628–6637. PMLR, 2019.

[45] Di Wang and Jinhui Xu. Escaping saddle points of empirical risk privately and scalably via dp-trust region method. In *Machine Learning and Knowledge Discovery in Databases: European Conference, ECML PKDD 2020, Ghent, Belgium, September 14–18, 2020, Proceedings, Part III*, pages 90–106. Springer, 2021.

[46] Di Wang, Minwei Ye, and Jinhui Xu. Differentially private empirical risk minimization revisited: Faster and more general. *Advances in Neural Information Processing Systems*, 30, 2017.

[47] Lingxiao Wang, Bargav Jayaraman, David Evans, and Quanquan Gu. Efficient privacy-preserving stochastic nonconvex optimization. In *Uncertainty in Artificial Intelligence*, pages 2203–2213. PMLR, 2023.

[48] Lingxiao Wang, Difan Zou, Kumar Kshitij Patel, Jingfeng Wu, and Nathan Srebro. Private overparameterized linear regression without suffering in high dimensions, 2024.

[49] Jingfeng Wu, Difan Zou, Zixiang Chen, Vladimir Braverman, Quanquan Gu, and Sham M Kakade. Finite-sample analysis of learning high-dimensional single relu neuron. 2023.

[50] Hanshen Xiao, Zihang Xiang, Di Wang, and Srinivas Devadas. A theory to instruct differentially-private learning via clipping bias reduction. In *2023 IEEE Symposium on Security and Privacy (SP)*, pages 2170–2189. IEEE Computer Society, 2023.

[51] Jiaqi Zhang, Kai Zheng, Wenlong Mou, and Liwei Wang. Efficient private erm for smooth objectives. *arXiv preprint arXiv:1703.09947*, 2017.

[52] Qiuchen Zhang, Jing Ma, Jian Lou, and Li Xiong. Private stochastic non-convex optimization with improved utility rates. In *Proceedings of the Thirtieth International Joint Conference on Artificial Intelligence*, 2021.

[53] Yingxue Zhou, Zhiwei Steven Wu, and Arindam Banerjee. Bypassing the ambient dimension: Private sgd with gradient subspace identification. *arXiv preprint arXiv:2007.03813*, 2020.

[54] Liyang Zhu, Meng Ding, Vaneet Aggarwal, Jinhui Xu, and Di Wang. Improved analysis of sparse linear regression in local differential privacy model. *arXiv preprint arXiv:2310.07367*, 2023.

[55] Liyang Zhu, Amina Manseur, Meng Ding, Jinyan Liu, Jinhui Xu, and Di Wang. Truthful high dimensional sparse linear regression. *arXiv preprint arXiv:2410.13046*, 2024.

[56] Difan Zou, Jingfeng Wu, Vladimir Braverman, Quanquan Gu, and Sham Kakade. Benign overfitting of constant-stepsize sgd for linear regression. In *Conference on Learning Theory*, pages 4633–4635. PMLR, 2021.

# A  Algorithms

---

**Algorithm 1** DP-GLMtron

---

1: **Input**: Training Samples: $\{(x_i, y_i)\}_{i=0}^{N-1}$, Estimating Samples: $\{(x_i', y_i')\}_{i=1}^{m}$, Learning Rate: $\eta$, DP Noise Multiplier: $f$, Expected $\mathbf{x}$ Norm: $\sqrt{\alpha \operatorname{tr}(\mathbf{H})}$, Privacy Parameters: $\varepsilon', \delta'$, Clipping list $\gamma = \{\}$

2: Randomly permute $\{(\mathbf{x}_i, \mathbf{y}_i)\}_{i=0}^{N-1}$

3: Initialize $\mathbf{w}_0 \to 0$

4: **for** $t = 0, \ldots, N-1$ **do**

5:     $s_t \leftarrow$ **DP-Threshold**$(\{(\mathbf{x}_i', y_i')\}_{i=1}^{m}, \mathbf{w}_t, \varepsilon', \delta')$

6:     $\gamma_t = \sqrt{2\alpha \operatorname{tr}(\mathbf{H})} C_2 \log^{2a} N \cdot s_t$

7:     Sample $\mathbf{g}_t \sim \mathcal{N}(0, \mathbf{I}_{d \times d})$

8:     $\mathbf{w}_{t+1} \leftarrow \mathbf{w}_t - \eta(\boldsymbol{l}_t + 2f\gamma_t \mathbf{g}_t)$

9:     where $\boldsymbol{l}_t = \operatorname{clip}_{\gamma_t}(\mathbf{x}_t^\top (\operatorname{ReLU}(\mathbf{x}_t^\top \mathbf{w}_t) - \mathbf{y}_t))$

10: **end for**

11: **return** $\mathbf{w} \leftarrow \frac{1}{N} \sum_{t=0}^{N-1} \mathbf{w}_t$

---

---

**Algorithm 2** DP-Threshold

---

1: **Input**: Estimating Samples: $\{(x_i', y_i')\}_{i=1}^{m}$, Privacy Parameters: $\varepsilon', \delta'$, Model $\mathbf{w}$, Distribution Parameters: $a, b_z$ in Definition 3.5

2: $\forall i \in [m], d_i^1 \leftarrow (\mathbf{x}_i^\top \mathbf{w} - y_i)^2, d_i^2 \leftarrow y_i^2, d_i^3 \leftarrow (\mathbf{x}_i^\top \mathbf{w})^2, \forall i \in [m]$

3: Partition $\{d_i^1\}_{i=1}^{m}$, $\{d_i^2\}_{i=1}^{m}$ and $\{d_i^3\}_{i=1}^{m}$ into three $K = \lceil C_1 \log(N/(\delta'))/\varepsilon' \rceil$ subsets of equal size separately, namely $\{D_j^1\}_{j=1}^{K}$, $\{D_j^2\}_{j=1}^{K}$ and $\{D_j^3\}_{j=1}^{K}$

4: **for** p=1,2,3 **do**

5:     $(d_j^p)' \leftarrow \sum_{i \in D_j^p} d_i^p / |D_j^p|, \forall j \in [K]$

6:     Partition $[0, \infty)$ into geometrically increasing intervals $\Omega^p := \{\ldots, [2^{-1}, 1), [1, 2), [2, 2^2), \ldots\} \cup \{[0, 0]\}$

7:     Compute $\overline{d}_k^p \leftarrow \sum_{j \in D_k^p} d_j^p \mathbb{1}(d_j^p \leq (d_k^p)')/|D_k^p|$, where $D_k^p$ is the interval in $\Omega^p$

8:     **if** $\overline{d}_k^p \in (0, 2\log(1/\delta')/(\varepsilon'K) + (1/K))$ **then**

9:         $\widetilde{\overline{d}}_k^p \leftarrow \overline{d}_k^p + p_k^p$, where $q_k^p \sim \operatorname{Lap}(0, 2/(\varepsilon'K))$

10:     **else**

11:         $\widetilde{\overline{d}}_k^p \leftarrow 0$

12:     **end if**

13:     Let $[s_1^p, s_2^p]$ be the nonempty interval containing the largest $\widetilde{\overline{d}}_k^p$

14:     **if** p=3 **then**

15:         set $s^p = \sqrt{2(s_2^p + \sigma^2 \cdot \log^{2a}(1/b_z))}$

16:     **else**

17:         set $s^p = \sqrt{2s_2^p}$

18:     **end if**

19: **end for**

20: **return** $s = \max\{s^p\}_{p=1}^{3}$

---

**Algorithm 3** DP-TAGLMtron

1: **Input**: Training Samples: $\{(x_i, y_i)\}_{i=0}^{N-1}$, Estimating Samples: $\{(x_i', y_i')\}_{i=1}^m$, Learning Rate: $\eta$, DP Noise Multiplier: $f$, Expected $\mathbf{x}$ Norm: $\sqrt{\alpha \operatorname{tr}(\mathbf{H})}$, Privacy Parameters: $\varepsilon', \delta'$, Clipping list $\boldsymbol{\gamma} = \{\}$
2: Initialize $w_0 \leftarrow \mathbf{0}$
3: **for** $t = 0 \dots N - 1$ **do**
4: $\quad s_t \leftarrow$ **DP-Threshold**$(\{(\mathbf{x}_i', y_i')\}_{i=1}^m, \mathbf{w}_t, \varepsilon', \delta')$
5: $\quad \gamma_t \leftarrow \sqrt{2\alpha \operatorname{tr}(\mathbf{H})} C_2 \log^{2a} N \cdot s_t$
6: $\quad$ add $\gamma_t$ to the list $\boldsymbol{\gamma}$ and $\Gamma = \max \boldsymbol{\gamma}$
7: $\quad \boldsymbol{l}_t \leftarrow \operatorname{clip}_{\gamma_t}(\mathbf{x}_t(\operatorname{ReLU}(\mathbf{x}_t^\top \mathbf{w}_t) - y_t))$
8: $\quad \boldsymbol{l}_{\leq t}' \leftarrow$ **DP-Tree-Aggregation**$(\boldsymbol{l}_{\leq t}, f, \Gamma)$
9: $\quad$ Update $\mathbf{w}_{t+1} \leftarrow \operatorname{argmin}_\mathbf{w} \langle \mathbf{w}, \boldsymbol{l}_t' \rangle + \frac{1}{2\eta} \|\mathbf{w}\|_2^2$
10: **end for**
11: **return** $\overline{\mathbf{w}}_N \leftarrow \frac{1}{N} \sum_{t=0}^{N-1} \mathbf{w}_t$

---

**Algorithm 4** DP-Tree-Aggregation

1: **Input**: Vectors: $\{\boldsymbol{l}_i\}_{i=0}^{t-1}$, Noise Paramters: $f, \Gamma$
2: **Initialization** Create a binary tree $\mathcal{T}$ of size $2^{\lceil \log_2 N \rceil + 1} - 1$ with $N$ leaf nodes, and each node is sampled i.i.d. from $\mathcal{N}(0, f^2 \Gamma^2 \mathbf{I}_{d \times d})$.
3: At $t$-th iteration, do:
4: Accept $\boldsymbol{l}_t$ and add it to all the nodes along the path to the root of $\mathcal{T}$.
5: Let $[e_1, \dots, e_h]$ be the list of nodes from the root of $\mathcal{T}$ to the $t$-th leaf node, with node $e_1$ being the root node and node $e_h$ being the leaf node, where $h$ is the height of tree.
6: Let $\boldsymbol{l}_{\leq t}' = \mathbf{0}^d$ and $\{b_1, \dots, b_h\}$ be the $h$ bit binary representation of $t$.
7: **for** $j \in [h]$ **do**
8: $\quad$ **if** $b_j = 1$ **then**
9: $\quad\quad \boldsymbol{l}_{\leq t}' \leftarrow \boldsymbol{l}_{\leq t}' + \boldsymbol{e}_j$, where $\boldsymbol{e}_j$ is the value in left sibling of $e_j$ node.
10: $\quad$ **end if**
11: **end for**
12: **return** $\boldsymbol{l}_{\leq t}'$

# B  Additional Experiment

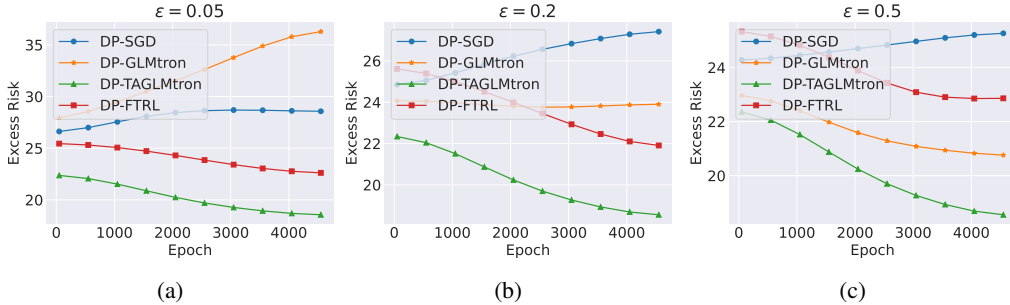

(a)                                    (b)                                    (c)

Figure 3: Comparative Performance of Various Differential Privacy Algorithms on MNIST Across Different Privacy Budgets. This figure illustrates the performance of four differential privacy algorithms—DP-SGD, DP-GLMtron, DP-TAGLMtron, and DP-FTRL—on generated Bernoulli distributed data, comparing excess risk across different sample sizes and privacy budgets ($\varepsilon = 0.05$, 0.2, 0.5).

In our second experiment, we evaluate the performance of four different differential privacy (DP) algorithms—DP-SGD, DP-GLMtron, DP-TAGLMtron, and DP-FTRL—using the MNIST dataset across varying sample sizes and privacy budgets denoted by $\varepsilon$, with values set at 0.05, 0.2, and 0.5. The learning rate was initially set to $0.05$, with $N$ representing the sample size, which varied from 0 to 1000 and increased in steps to 100. We utilized a ReLU regression model with noise to be trained using the previously mentioned algorithms, integrating privacy-preserving noise adjustments. The primary metric for evaluation was the average excess risk across varying dataset sizes and $\varepsilon$ values.

The key insights from our findings are: Under Tight Privacy Constraints ($\varepsilon = 0.05$): DP-TAGLMtron demonstrated superior performance across all sample sizes, achieving the lowest excess risk, while DP-GLMtron's excess risk increased with larger sample sizes. Under Moderate Privacy Constraints ($\varepsilon = 0.2$): The performance differential between our DP-GLMtron and the baseline narrowed, and DP-TAGLMtron continued to exhibit lower excess risks. DP-SGD's performance suffered with larger sample sizes, highlighting its sensitivity to sample size under moderate privacy levels. Under Relaxed Privacy Constraints ($\varepsilon = 0.5$): Our proposed methods markedly outperformed both DP-SGD and DP-FTRL, showcasing their enhanced risk mitigation capabilities when privacy constraints are loosened. These insights demonstrate the effectiveness of our proposed algorithms, DP-TAGLMtron and DP-GLMtron, in optimizing the balance between privacy protection and model accuracy.

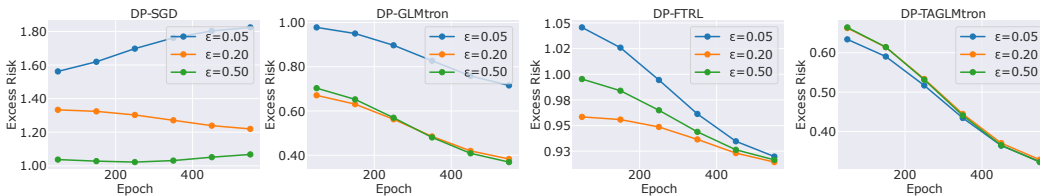

Figure 4: Performance of Different Algorithms under Various Privacy Budgets.

**Sensitivity of Privacy Budgets.** Here, we also included additional experimental results related to the performance of different algorithms under various privacy budgets. we demonstrate the sensitivity of the privacy budget for each algorithm, and it is clear that DP-SGD faces the same issue. When $\varepsilon$ is small, the tree mechanism in DP-FTRL allows it to add less noise compared to the naive Gradient Descent or GLMtron algorithm, resulting in similar performance between DP-FTRL and DP-GLMtron. Additionally, it is important to note that DP-FTRL should be compared with DP-TAGLMtron rather than DP-GLMtron, as both DP-FTRL and DP-TAGLMtron utilize the tree mechanism, whereas DP-GLMtron does not. Moreover, even though DP-FTRL converges at the end of the training, it can be observed that the excess risk for DP-FTRL is significantly larger than that of DP-TAGLMtron, indicating that it may converge to a suboptimal solution.

## C  Supported Lemmas

Here, we provide some relaxation for Assumption 3.6.

**Assumption C.1** (Moment symmetricity conditions [49]). Assume that:

1. For every $\mathbf{u} \in \mathbb{H}$, it holds that
$$\mathbb{E}[\mathbf{x}\mathbf{x}^\top \cdot \mathbb{1}[\mathbf{x}^\top\mathbf{u} > 0]] = \mathbb{E}[\mathbf{x}\mathbf{x}^\top \cdot \mathbb{1}[\mathbf{x}^\top\mathbf{u} < 0]].$$

2. For every $\mathbf{u} \in \mathbb{H}$ and $\mathbf{v} \in \mathbb{H}$, it holds that
$$\mathbb{E}[\mathbf{x}\mathbf{x}^\top \cdot \mathbb{1}[\mathbf{x}^\top\mathbf{u} > 0, \mathbf{x}^\top\mathbf{v} > 0]] = \mathbb{E}[\mathbf{x}\mathbf{x}^\top \cdot \mathbb{1}[\mathbf{x}^\top\mathbf{u} < 0, \mathbf{x}^\top\mathbf{v} < 0]].$$

3. For every $\mathbf{u} \in \mathbb{H}$, it holds that
$$\mathbb{E}[\mathbf{x}^{\otimes 4} \cdot \mathbb{1}[\mathbf{x}^\top\mathbf{u} > 0]] = \mathbb{E}[\mathbf{x}^{\otimes 4} \cdot \mathbb{1}[\mathbf{x}^\top\mathbf{u} < 0]].$$

4. For every $\mathbf{u} \in \mathbb{H}$ and $\mathbf{v} \in \mathbb{H}$, it holds that
$$\mathbb{E}[(\mathbf{x}^\top\mathbf{v})^2\mathbf{x}\mathbf{x}^\top \cdot \mathbb{1}[\mathbf{x}^\top\mathbf{u} > 0, \mathbf{x}^\top\mathbf{v} > 0]] = \mathbb{E}[(\mathbf{x}^\top\mathbf{v})^2\mathbf{x}\mathbf{x}^\top \cdot \mathbb{1}[\mathbf{x}^\top\mathbf{u} < 0, \mathbf{x}^\top\mathbf{v} < 0]].$$

**Lemma C.2** ([49]). *The following results are direct consequences of Assumption C.1.*

1. *Under Assumption C.1, it holds that: for every vector $\mathbf{u} \in \mathbb{H}$,*
$$\mathbb{E}[\mathbf{x}\mathbf{x}^\top \cdot \mathbb{1}[\mathbf{x}^\top\mathbf{u} > 0]] = \frac{1}{2} \cdot \mathbb{E}[\mathbf{x}\mathbf{x}^\top] =: \frac{1}{2} \cdot \mathbf{H}.$$

2. *Under Assumption C.1, it holds that: for every vector $\mathbf{u} \in \mathbb{H}$,*
$$\mathbb{E}[\mathbf{x}^{\otimes 4} \cdot \mathbb{1}[\mathbf{x}^\top\mathbf{u} > 0]] = \frac{1}{2} \cdot \mathbb{E}[\mathbf{x}^{\otimes 4}] =: \frac{1}{2} \cdot \mathcal{M}$$

## D  DP-GLMtron

### D.1  Privacy Guarantee

To guarantee the privacy of Algorithm 1, we should first ensure that each step of DP-GLMtron is private.

**Lemma D.1.** *Each update step of DP-GLMtron (Algorithm 1) ensures $(\varepsilon_0, \delta_0)$-differential privacy, provided that $f = \frac{c}{\varepsilon_0}$, where $c \geq \sqrt{2\log(1.25/\delta_0)}$, and $\gamma_t$ denotes the clipping norm.*

**Proof.** We first consider $\{\mathbf{w}_t\}_{t=0}^{N-1}$.

Each update step (excluding the DP-noise addition) is of the form:
$$\mathbf{w}_{t+1} \leftarrow \mathbf{w}_t - \eta \, \mathrm{clip}_{\gamma_t}(\mathbf{x}_t(\mathrm{ReLU}(\mathbf{x}_t^\top\mathbf{w}_t) - y_t)),$$

where $\mathrm{clip}_{\gamma_t}(\mathbf{v}) = \mathbf{v} \cdot \max\{1, \frac{\gamma_t}{\|\mathbf{v}\|_2}\}$. Consequently, the local $L_2$ sensitivity of $\mathbf{w}_{t+1}$ is determined by analyzing the variation in the $t^{\text{th}}$ iteration data sample, as follows:

$$\begin{aligned}
\Delta_2 &= \|\mathbf{w}_{t+1}' - \mathbf{w}_{t+1}\| \\
&= \|\eta \, \mathrm{clip}_{\gamma_t}(\mathbf{x}_t'(\mathrm{ReLU}(\mathbf{x}_t'^\top\mathbf{w}_t) - y_t')) - \eta \, \mathrm{clip}_{\gamma_t}(\mathbf{x}_t(\mathrm{ReLU}(\mathbf{x}_t^\top\mathbf{w}_t) - y_t))\| \\
&\leq 2\eta \|\mathrm{clip}_{\gamma_t}(\mathbf{x}_t(\mathrm{ReLU}(\mathbf{x}_t^\top\mathbf{w}_t) - y_t))\| \\
&= 2\eta\gamma_t.
\end{aligned}$$

Moreover, denoting $\varepsilon' = \varepsilon_0/\sqrt{8N\log(2/\delta_0)}$ and $\delta' = \delta_0/(2N)$, according to Lemma 2.3 in [24], we could know that $\{s_t\}_{t=0}^N$ is $(\varepsilon', \delta')$-DP.

$\square$

**Lemma D.2** ([17]). *For a domain $\mathcal{D}$, let $\mathcal{R}^{(i)} : f \times \mathcal{D} \to \mathcal{S}^{(i)}$ for $i \in [n]$ (where $\mathcal{S}^{(i)}$ is the range space of $\mathcal{R}^{(i)}$) be a sequence of algorithms such that $\mathcal{R}^{(i)}(z_{1:i-1}, \cdot)$ is a $(\varepsilon_0, \delta_0)$-DP local randomizer*

*for all values of auxiliary inputs $z_{1:i-1} \in \mathcal{S}^{(1)} \times \cdots \times \mathcal{S}^{(i-1)}$. Let $\mathcal{A}_s : \mathcal{D}^n \to \mathcal{S}^{(1)} \times \cdots \times \mathcal{S}^{(n)}$ be the algorithm that given a dataset $x_{1:n} \in \mathcal{D}^n$, samples a uniformly random permutation $\pi$, then sequentially computes $z_i = \mathcal{R}^{(i)}(z_{1:i-1}, x_{\pi(i)})$ for $i \in [n]$, and outputs $z_{1:n}$. Then for any $\delta \in [0, 1]$ such that $\varepsilon_0 \le \log(\frac{n}{16 \log(2/\delta)})$, $\mathcal{A}_s$ is $(\varepsilon, \delta + O(e^\varepsilon \delta_0 n))$-DP where $\varepsilon$ is:*

$$\varepsilon = O\left((1 - e^{-\varepsilon_0})\left(\frac{\sqrt{e^{\varepsilon_0} \log(1/\delta)}}{\sqrt{n}} + \frac{e^{\varepsilon_0}}{n}\right)\right).$$

We are now prepared to prove Theorem 4.1, utilizing the lemmas discussed above.

Firstly, we reformulate the update rule into a sequence of one-step algorithms as follows:

$$\mathcal{R}^{(t+1)}(u_{0:t}, (\mathbf{x}, y)) := \mathbf{w}_{t+1} \leftarrow \mathbf{w}_t(u_{0:t}) - \eta \operatorname{clip}_{\gamma_t}(\mathbf{x}_{\pi(t)}(\operatorname{ReLU}(\mathbf{x}_{\pi(t)}^\top \mathbf{w}_t(u_{0:t})) - y_{\pi(t)})) - 2\eta \gamma_t f g_t,$$

where $u$ denotes auxiliary inputs, and $\pi(t)$ represents the sample at the $t$-th iteration after randomly permuting the input data.

From Lemma D.1, each $\mathcal{R}^{(t+1)}(u_{0:t}, \cdot)$ is a $(\varepsilon_0, \delta_0)$-DP local randomizer algorithm, where $\varepsilon_0 \le \log(\frac{N}{16 \log(2/\widehat{\delta})})$. The output of DP-GLMtron is derived through post-processing of the shuffled outputs $u_{t+1} = \mathcal{R}^{(t+1)}(u_{0:t}, (\mathbf{x}, y))$ for $t \in 0, \ldots, N - 1$. Therefore, by Lemma D.2, Algorithm DP-GLMtron adheres to $(\widehat{\varepsilon}, \widehat{\delta} + O(e^{\widehat{\varepsilon}} \delta_0 N))$-DP, where:

$$\widehat{\varepsilon} = O\left((1 - e^{-\varepsilon_0})\left(\frac{\sqrt{e^{\varepsilon_0} \log(1/\widehat{\delta})}}{\sqrt{N}} + \frac{e^{\varepsilon_0}}{N}\right)\right).$$

Assuming $\varepsilon_0 \le \frac{1}{2}$, we can infer the existence of some constant $c_1 > 0$ such that:

$$\widehat{\varepsilon} \le c_1 \cdot (1 - e^{-\varepsilon_0})\left(\frac{\sqrt{e^{\varepsilon_0} \log(1/\widehat{\delta})}}{\sqrt{N}} + \frac{e^{\varepsilon_0}}{N}\right)$$

$$\le c_1 \cdot \left((e^{\varepsilon_0/2} - e^{-\varepsilon_0/2})\sqrt{\frac{\log(1/\widehat{\delta})}{N}} + (e^{\varepsilon_0} - 1)\frac{1}{N}\right)$$

$$\le c_1 \cdot \left(((1 + \varepsilon_0) - (1 - \varepsilon_0/2))\sqrt{\frac{\log(1/\widehat{\delta})}{N}} + ((1 + 2\varepsilon_0) - 1)\frac{1}{N}\right) \tag{4}$$

$$= c_1 \cdot \varepsilon_0\left(\frac{1}{2}\sqrt{\frac{\log(1/\widehat{\delta})}{N}} + \frac{2}{N}\right).$$

By setting $f = \frac{\sqrt{2 \log(1.25/\delta_0)}}{\varepsilon_0}$ in Lemma D.1, we ensure that each update step of DP-GLMtron independently satisfies $(\varepsilon_0, \delta_0)$-DP, based on standard Gaussian mechanism. Replacing $\varepsilon_0 = \frac{\sqrt{2 \log(1.25/\delta_0)}}{f}$, we obtain:

$$\widehat{\varepsilon} \le c_1 \cdot \frac{\sqrt{2 \log(1.25/\delta_0) \log(1/\widehat{\delta})}}{f\sqrt{N}} \tag{5}$$

To satisfy overall $(\varepsilon, \delta)$-DP, set $\widehat{\delta} = \frac{\delta}{2}$, and $\delta_0 = c_2 \cdot \frac{\delta}{e^{\widehat{\varepsilon}} N}$ for some constant $c_2 > 0$. From this, we have:

$$\widehat{\varepsilon} \le c_1 \cdot \frac{\sqrt{2 \log(c_2 \cdot 1.25 \cdot e^{\widehat{\varepsilon}} N/\delta) \cdot \log(2/\delta)}}{f\sqrt{N}} \tag{6}$$

For any $\varepsilon \le 1$, setting $f = c_3 \cdot \frac{\log(N/\delta)}{\varepsilon\sqrt{N}} \ge c_3 \frac{\sqrt{\log(N/\delta) \log(1/\delta)}}{\varepsilon\sqrt{N}}$ for a sufficiently large $c_3 > 0$ ensures that $\widehat{\varepsilon} \le \varepsilon$. Additionally, to fulfill Lemma D.2's assumption, $\varepsilon_0 < \frac{1}{2}$ must be satisfied, which is attainable by setting $\varepsilon = O(\sqrt{\frac{\log(N/\delta)}{N}})$.

This implies that for $f = \Omega(\frac{\log(N/\delta)}{\varepsilon\sqrt{N}})$, DP-GLMtron achieves $(\varepsilon, \delta)$-DP as long as $\varepsilon = O(\sqrt{\frac{\log(N/\delta)}{N}})$, thereby completing the proof.

## D.2 Utility Guarantee

Here we first provide several results that will be used in our DP-GLMtron utility bound.

**Lemma D.3** ([28]). *Let $m = \Omega(b\log(b/\delta')\log N/\varepsilon')$ and denote $V = \max\{\|\mathbf{w}^* - \mathbf{w}_t\|_{\mathbf{H}}^2 + \sigma^2, \|\mathbf{w}^*\|_{\mathbf{H}}^2 + \sigma^2, \|\mathbf{w}_t\|_{\mathbf{H}}^2 + \sigma^2\}$, with probability at least $1 - 1/b$, Algorithm 2 returns $s_t$ such that $V \leq \gamma_t^2 \leq 2V$.*

**Proof.** Consider each sample $\mathbf{x}_i \sim \mathcal{D}$ satisfying $(\mathbf{H}, C_2, a, b)$-Tail and the inherent noise $z_i$ satisfying $(\sigma^2, C_2, a, b)$-Tail. It implies the following facts with Assumption 3.4

- $\exists a > 0$ s.t. with probability $\geq 1 - b_{\mathbf{x}}$, such that $\|\mathbf{x}\|_2^2 \leq \alpha\,\mathrm{tr}(\mathbf{H}) \cdot \log^{2a}(1/b_{\mathbf{x}})$.

- $\exists a > 0$ s.t. with probability $\geq 1 - b_v$, such that $\langle \mathbf{x}, \mathbf{v}\rangle^2 \leq C_2^2 \mathbf{v}^\top \mathbf{H} \mathbf{v} \log(1/b_v)$.

- $\exists a > 0$ s.t. with probability $\geq 1 - b_z$, such that $\|z\|_2^2 \leq \sigma^2 \cdot \log^{2a}(1/b_z)$.

Additionally, we know that

$$\|\mathbf{x}_t(\mathrm{ReLU}(\mathbf{x}_t^\top\mathbf{w}_t) - y_t)\| = \|\mathbf{x}_t(\max(0, \mathbf{x}_t^\top\mathbf{w}_t) - y_t)\| \leq \|\mathbf{x}_t\|\|(\max(0, \mathbf{x}_t^\top\mathbf{w}_t) - y_t)\|$$

According to the aforementioned facts, it holds that with probability $\geq 1 - b_{\mathbf{x}}$, such that

$$\|\mathbf{x}\|_2^2 \leq \alpha\,\mathrm{tr}(\mathbf{H}) \cdot \log^{2a}(1/b_{\mathbf{x}}).$$

Moreover,

$$\sqrt{s_t}\| \leq \max\{\sqrt{2((\mathbf{x}_t^\top(\mathbf{w}_t - \mathbf{w}^*))^2 + z_t^2)}, \sqrt{2((\mathbf{x}_t^\top\mathbf{w}_*)^2 + z_t^2)}, \sqrt{2((\mathbf{x}_t^\top\mathbf{w}_t)^2 + z_t^2)}\}$$

$$\leq \sqrt{2C_2^2 \log^{2a}(1/b)} \cdot \max\{\sqrt{\|\mathbf{w}^* - \mathbf{w}_t\|_{\mathbf{H}}^2 + \sigma^2}, \sqrt{\|\mathbf{w}^*\|_{\mathbf{H}}^2 + \sigma^2}, \sqrt{\|\mathbf{w}_t\|_{\mathbf{H}}^2 + \sigma^2}\}.$$

The second part of the proof can be directly derived from Lemma 2.3 in [24] and Theorem 5 in [28], because each part of $d_i^p$, with high probability, holds that

$$\sqrt{s_t}\| \geq \max\{\sqrt{\|\mathbf{w}^* - \mathbf{w}_t\|_{\mathbf{H}}^2 + \sigma^2}, \sqrt{\|\mathbf{w}^*\|_{\mathbf{H}}^2 + \sigma^2}, \sqrt{\|\mathbf{w}_t\|_{\mathbf{H}}^2 + \sigma^2}\}.$$

$\square$

Therefore, we have the following clipping results:

$$\|\mathbf{x}_t(\mathrm{ReLU}(\mathbf{x}_t^\top\mathbf{w}_t) - y_t)\|_2^2$$
$$\leq 2\alpha\,\mathrm{tr}(\mathbf{H}) \cdot \log^{2a}(1/b_{\mathbf{x}}) \cdot C_2^2 \log^{2a}(1/b) \cdot \max\{\|\mathbf{w}^* - \mathbf{w}_t\|_{\mathbf{H}}^2 + \sigma^2, \|\mathbf{w}^*\|_{\mathbf{H}}^2 + \sigma^2, \|\mathbf{w}_t\|_{\mathbf{H}}^2 + \sigma^2\}$$
$$\leq 2\alpha\,\mathrm{tr}(\mathbf{H}) \cdot \log^{4a}(1/b) \cdot C_2^2 \cdot s_t^2.$$

**Lemma D.4** (Generic bounds on the DP-GLMtron iterates [49]). *Suppose that Assumption 3.6 holds. Considering the DP-GLMtron algorithm, we have the following recursion:*

- $\mathbf{A}_t \preceq \mathbf{A}_{t-1} - \frac{\eta}{2}(\mathbf{H}\mathbf{A}_{t-1} + \mathbf{A}_{t-1}\mathbf{H}) + \eta^2\mathcal{M} \circ \mathbf{A}_{t-1} + \eta^2\sigma^2\mathbf{H} + 4\eta^2\Gamma^2 f^2\mathbf{I}$

- $\mathbf{A}_t \succeq \mathbf{A}_{t-1} - \frac{\eta}{2}(\mathbf{H}\mathbf{A}_{t-1} + \mathbf{A}_{t-1}\mathbf{H}) + \frac{\eta^2}{4}\mathcal{M} \circ \mathbf{A}_{t-1} + \eta^2\sigma^2\mathbf{H} + 4\eta^2\widetilde{\Gamma}^2 f^2\mathbf{I}$

*where $\mathbf{A}_t := \mathbb{E}(\mathbf{w}_t - \mathbf{w}_*)(\mathbf{w}_t - \mathbf{w}_*)^\top$, $t \geq 0$*

Now consider the recursion of $\mathbf{A}_t$ given in Lemma D.4. Note that $\mathbf{A}_t$ is related to $\mathbf{A}_{t-1}$ through a linear operator, therefore $\mathbf{A}_t$ can be understood as the sum of two iterates, i.e., $\mathbf{A}_t := \mathbf{B}_t + \mathbf{C}_t$, where

$$\begin{cases} \mathbf{B}_t \preceq (\mathcal{I} - \frac{\eta}{2} \cdot \mathcal{T}(2\eta)) \circ \mathbf{B}_{t-1}; \\ \mathbf{B}_0 = (\mathbf{w}_0 - \mathbf{w}_*)^{\otimes 2} \end{cases} \quad \begin{cases} \mathbf{C}_t \preceq (\mathcal{I} - \frac{\eta}{2} \cdot \mathcal{T}(2\eta)) \circ \mathbf{C}_{t-1} + \eta^2\sigma^2\mathbf{H} + 4\eta^2\Gamma^2 f^2\mathbf{I}; \\ \mathbf{C}_0 = 0 \end{cases}$$

$$(7)$$

and

$$\begin{cases} \mathbf{B}_t \succeq (\mathcal{I} - \frac{\eta}{2} \cdot \mathcal{T}(\frac{\eta}{2})) \circ \mathbf{B}_{t-1}; \\ \mathbf{B}_0 = (\mathbf{w}_0 - \mathbf{w}_*)^{\otimes 2} \end{cases} \quad \begin{cases} \mathbf{C}_t \succeq (\mathcal{I} - \frac{\eta}{2} \cdot \mathcal{T}(\frac{\eta}{2})) \circ \mathbf{C}_{t-1} + \eta^2\sigma^2\mathbf{H} + 4\eta^2\widetilde{\Gamma}^2 f^2\mathbf{I}; \\ \mathbf{C}_0 = 0 \end{cases}$$

$$(8)$$

where
$$\begin{cases} (\mathcal{I} - \frac{\eta}{2} \cdot \mathcal{T}(2\eta)) \circ \mathbf{A}_{t-1} := \mathbf{A}_{t-1} - \frac{\eta}{2}(\mathbf{H}\mathbf{A}_{t-1} + \mathbf{A}_{t-1}\mathbf{H}) + \eta^2 \mathcal{M} \circ \mathbf{A}_{t-1}; \\ (\mathcal{I} - \frac{\eta}{2} \cdot \mathcal{T}(\frac{\eta}{2})) \circ \mathbf{A}_{t-1} := \mathbf{A}_{t-1} - \frac{\eta}{2}(\mathbf{H}\mathbf{A}_{t-1} + \mathbf{A}_{t-1}\mathbf{H}) + \frac{\eta^2}{4} \mathcal{M} \circ \mathbf{A}_{t-1} \end{cases}$$

Besides, since our DP-GLM-tron is run with constant stepsize $\eta$ and outputs the average of the iterates:

$$\overline{\mathbf{w}}_N := \frac{1}{N} \sum_{t=0}^{N-1} \mathbf{w}_t. \tag{9}$$

Then, the following lemma holds:

**Lemma D.5.** *Suppose that Assumption 3.6 hold. For $\overline{\mathbf{w}}_N$ defined in Equation* (9)*, we have that*

$$\mathbb{E}\langle \mathbf{H}, (\overline{\mathbf{w}}_N - \mathbf{w}_*)^{\otimes 2} \rangle \leq \sum_{t=0}^{N-1} \sum_{k=t}^{N-1} \frac{1}{\eta N^2} \langle (\mathbf{I} - \frac{\eta}{2}\mathbf{H})^{k-t}\mathbf{H}, \mathbf{A}_t \rangle$$

$$\mathbb{E}\langle \mathbf{H}, (\overline{\mathbf{w}}_N - \mathbf{w}_*)^{\otimes 2} \rangle \geq \sum_{t=0}^{N-1} \sum_{k=t}^{N-1} \frac{1}{2\eta N^2} \langle (\mathbf{I} - \frac{\eta}{2}\mathbf{H})^{k-t}\mathbf{H}, \mathbf{A}_t \rangle.$$

**Proof.**
$$\begin{aligned} \mathbb{E}[\mathbf{w}_t - \mathbf{w}_* \mid \mathbf{w}_{t-1}] =& \mathbb{E}[(\mathbf{I} - \eta \mathbb{1}[\mathbf{x}_t^\top \mathbf{w}_{t-1} > 0]\mathbf{x}_t\mathbf{x}_t^\top)(\mathbf{w}_{t-1} - \mathbf{w}_*) \mid \mathbf{w}_{t-1}] + 2\eta \Gamma f \mathbb{E}[\mathbf{g}_{t-1} \mid \mathbf{w}_{t-1}] \\ &+ \eta \cdot \mathbb{E}[(\mathbb{1}[\mathbf{x}_t^\top \mathbf{w}_* > 0] - \mathbb{1}[\mathbf{x}_t^\top \mathbf{w}_{t-1} > 0])\mathbf{x}_t\mathbf{x}_t^\top \mathbf{w}_* \mid \mathbf{w}_{t-1}] + \eta \mathbb{E}[z_t \mathbf{x}_t \mid \mathbf{w}_{t-1}] \\ =& \mathbb{E}[(\mathbf{I} - \eta \mathbb{1}[\mathbf{x}_t^\top \mathbf{w}_{t-1} > 0]\mathbf{x}_t\mathbf{x}_t^\top)(\mathbf{w}_{t-1} - \mathbf{w}_*) \mid \mathbf{w}_{t-1}] \\ =& (\mathbf{I} - \frac{\eta}{2}\mathbf{H})(\mathbf{w}_{t-1} - \mathbf{w}_*) \end{aligned}$$

The remaining proof simply follows [56]. $\square$

From the decomposition presented in Equation (7) and Equation (8), we know that $\sum_{t=0}^{N} \mathbf{A}_t = \sum_{t=0}^{N} \mathbf{B}_t + \sum_{t=0}^{N} \mathbf{C}_t$. With this foundation, we can now bound the bias and variance terms separately.

**Upper bound for variance error**

For $t = 0$ we have $\mathbf{C}_0 = 0 \preceq \frac{\eta\sigma^2}{1-\eta(\alpha\operatorname{tr}(\mathbf{H}))}\mathbf{I} + \frac{4\eta\Gamma^2 f^2}{1-\eta(\alpha\operatorname{tr}(\mathbf{H}))}\mathbf{H}^{-1}$.

We then assume that $\mathbf{C}_{t-1} \preceq \frac{\eta\sigma^2}{1-\eta(\alpha\operatorname{tr}(\mathbf{H}))}\mathbf{I} + \frac{4\eta\Gamma^2 f^2}{1-\eta(\alpha\operatorname{tr}(\mathbf{H}))}\mathbf{H}^{-1}$, and exam $\mathbf{C}_t$ based on Equation (7):

$$\begin{aligned} \mathbf{C}_t &\preceq (\mathcal{I} - \eta \cdot \mathcal{T}(\eta)) \circ \mathbf{C}_{t-1} + \eta^2\sigma^2\mathbf{H} + 4\eta^2\Gamma^2 f^2\mathbf{I} \\ &= \mathbf{C}_{t-1} - \frac{\eta}{2}(\mathbf{H}\mathbf{C}_{t-1} + \mathbf{C}_{t-1}\mathbf{H}) + \eta^2\mathcal{M} \circ \mathbf{C}_{t-1} + \eta^2\sigma^2\mathbf{H} + 4\eta^2\Gamma^2 f^2\mathbf{I} \\ &\preceq \frac{\eta\sigma^2}{1-\eta(\alpha\operatorname{tr}(\mathbf{H}))}\mathbf{I} + \frac{4\eta\Gamma^2 f^2}{1-\eta(\alpha\operatorname{tr}(\mathbf{H}))}\mathbf{H}^{-1} - \eta\left(\frac{\eta\sigma^2}{1-\eta(\alpha\operatorname{tr}(\mathbf{H}))}\mathbf{H} + \frac{4\eta\Gamma^2 f^2}{1-\eta(\alpha\operatorname{tr}(\mathbf{H}))}\mathbf{I}\right) \\ &\quad + \eta^2(\alpha\operatorname{tr}(\mathbf{H}))\left(\frac{\eta\sigma^2}{1-\eta(\alpha\operatorname{tr}(\mathbf{H}))}\mathbf{H} + \frac{4\eta\Gamma^2 f^2}{1-\eta(\alpha\operatorname{tr}(\mathbf{H}))}\mathbf{I}\right) + \eta^2\sigma^2\mathbf{H} + 4\eta^2\Gamma^2 f^2\mathbf{I} \\ &\preceq \frac{\eta\sigma^2}{1-\eta(\alpha\operatorname{tr}(\mathbf{H}))}\mathbf{I} + \frac{4\eta\Gamma^2 f^2}{1-\eta(\alpha\operatorname{tr}(\mathbf{H}))}\mathbf{H}^{-1}. \end{aligned}$$

For the simplicity, we define $\boldsymbol{\Sigma} := \sigma^2\mathbf{H} + 4\Gamma^2 f^2\mathbf{I}$. By the definitions of $\mathcal{T}$ and $\widetilde{\mathcal{T}}$, we have:

$$\begin{aligned} \mathbf{C}_t &= (\mathcal{I} - \frac{\eta}{2} \cdot \mathcal{T}(2\eta)) \circ \mathbf{C}_{t-1} + \eta^2\boldsymbol{\Sigma} \\ &= (\mathcal{I} - \frac{\eta}{2} \cdot \widetilde{\mathcal{T}}(2\eta)) \circ \mathbf{C}_{t-1} + \eta^2(\mathcal{M} - \frac{1}{4}\widetilde{\mathcal{M}}) \circ \mathbf{C}_{t-1} + \eta^2\boldsymbol{\Sigma} \\ &\preceq (\mathcal{I} - \frac{\eta}{2} \cdot \widetilde{\mathcal{T}}(2\eta)) \circ \mathbf{C}_{t-1} + \eta^2\mathcal{M} \circ \mathbf{C}_{t-1} + \eta^2\boldsymbol{\Sigma}, \end{aligned}$$

where the last inequality is due to the fact that $\widetilde{\mathcal{M}}$ is a PSD mapping. Then by the iteration of variance, we have for all $t \geq 0$,

$$\mathcal{M} \circ \mathbf{C}_t \preceq \mathcal{M} \circ \left( \frac{\eta \sigma^2}{1 - \eta(\alpha \operatorname{tr}(\mathbf{H}))} \mathbf{I} + \frac{4\eta \Gamma^2 f^2}{1 - \eta(\alpha \operatorname{tr}(\mathbf{H}))} \mathbf{H}^{-1} \right) \preceq \frac{\eta \sigma^2 (\alpha \operatorname{tr}(\mathbf{H}))}{1 - \eta(\alpha \operatorname{tr}(\mathbf{H}))} \mathbf{H} + \frac{4\eta \Gamma^2 f^2 (\alpha \operatorname{tr}(\mathbf{H}))}{1 - \eta(\alpha \operatorname{tr}(\mathbf{H}))} \mathbf{I}$$

Substituting above into Appendix D.2, we obtain:

$$
\begin{aligned}
\mathbf{C}_t &\preceq \left( \mathcal{I} - \frac{\eta}{2} \cdot \widetilde{\mathcal{T}}(2\eta) \right) \circ \mathbf{C}_{t-1} + \frac{\eta^3 (\alpha \operatorname{tr}(\mathbf{H}))}{1 - \eta(\alpha \operatorname{tr}(\mathbf{H}))} \cdot (\sigma^2 \mathbf{H} + 4\Gamma^2 f^2 \mathbf{I}) + \eta^2 \mathbf{\Sigma} \\
&= \left( \mathcal{I} - \frac{\eta}{2} \cdot \widetilde{\mathcal{T}}(2\eta) \right) \circ \mathbf{C}_{t-1} + \frac{\eta^3 (\alpha \operatorname{tr}(\mathbf{H}))}{1 - \eta(\alpha \operatorname{tr}(\mathbf{H}))} \cdot (\sigma^2 \mathbf{H} + 4\Gamma^2 f^2 \mathbf{I}) + \eta^2 (\sigma^2 \mathbf{H} + 4\Gamma^2 f^2 \mathbf{I}) \\
&= \left( \mathcal{I} - \frac{\eta}{2} \cdot \widetilde{\mathcal{T}}(2\eta) \right) \circ \mathbf{C}_{t-1} + \frac{\eta^2}{1 - \eta(\alpha \operatorname{tr}(\mathbf{H}))} \cdot (\sigma^2 \mathbf{H} + 4\Gamma^2 f^2 \mathbf{I}) \\
&\preceq \frac{\eta^2}{1 - \eta(\alpha \operatorname{tr}(\mathbf{H}))} \cdot \sum_{k=0}^{t-1} \left( \mathcal{I} - \frac{\eta}{2} \cdot \widetilde{\mathcal{T}}(2\eta) \right)^k \circ (\sigma^2 \mathbf{H} + 4\Gamma^2 f^2 \mathbf{I}) \qquad (10) \\
&= \frac{\eta^2}{1 - \eta(\alpha \operatorname{tr}(\mathbf{H}))} \cdot \sum_{k=0}^{t-1} \left( \mathbf{I} - \frac{\eta}{2} \mathbf{H} \right)^k (\sigma^2 \mathbf{H} + 4\Gamma^2 f^2 \mathbf{I}) \left( \mathbf{I} - \frac{\eta}{2} \mathbf{H} \right)^k \\
&\preceq \frac{\eta^2}{1 - \eta(\alpha \operatorname{tr}(\mathbf{H}))} \cdot \sum_{k=0}^{t-1} \left( \mathbf{I} - \frac{\eta}{2} \mathbf{H} \right)^k (\sigma^2 \mathbf{H} + 4\Gamma^2 f^2 \mathbf{I}) \\
&= \frac{\eta \sigma^2}{1 - \eta(\alpha \operatorname{tr}(\mathbf{H}))} \cdot \left( \mathbf{I} - \left( \mathbf{I} - \frac{\eta}{2} \mathbf{H} \right)^t \right) + \frac{4\eta \Gamma^2 f^2}{1 - \eta(\alpha \operatorname{tr}(\mathbf{H}))} \cdot \left( \mathbf{I} - \left( \mathbf{I} - \frac{\eta}{2} \mathbf{H} \right)^t \right) \cdot \mathbf{H}^{-1}
\end{aligned}
$$

Consequently, the variance error can be represented as follows, in accordance with Lemma D.5:

$$
\begin{aligned}
\text{Variance error} &\leq \frac{1}{\eta N^2} \left\langle \mathbf{I} - \left( \mathbf{I} - \frac{\eta}{2} \mathbf{H} \right)^N, \sum_{t=0}^{N} \mathbf{C}_t \right\rangle \\
&\leq \frac{1}{\eta N^2} \left\langle \mathbf{I} - \left( \mathbf{I} - \frac{\eta}{2} \mathbf{H} \right)^N, \sum_{t=0}^{N} \frac{\eta \sigma^2}{1 - \eta(\alpha \operatorname{tr}(\mathbf{H}))} \cdot \left( \mathbf{I} - \left( \mathbf{I} - \frac{\eta}{2} \mathbf{H} \right)^t \right) \right\rangle \\
&\quad + \frac{1}{\eta N^2} \left\langle \mathbf{I} - \left( \mathbf{I} - \frac{\eta}{2} \mathbf{H} \right)^N, \sum_{t=0}^{N} \frac{4\eta \Gamma^2 f^2}{1 - \eta(\alpha \operatorname{tr}(\mathbf{H}))} \cdot \left( \mathbf{I} - \left( \mathbf{I} - \frac{\eta}{2} \mathbf{H} \right)^t \right) \cdot \mathbf{H}^{-1} \right\rangle \quad (11) \\
&\leq \frac{\sigma^2}{(1 - \eta(\alpha \operatorname{tr}(\mathbf{H}))) N} \left\langle \mathbf{I} - \left( \mathbf{I} - \frac{\eta}{2} \mathbf{H} \right)^N, \left( \mathbf{I} - \left( \mathbf{I} - \frac{\eta}{2} \mathbf{H} \right)^N \right) \right\rangle \\
&\quad + \frac{4\Gamma^2 f^2}{(1 - \eta(\alpha \operatorname{tr}(\mathbf{H}))) N} \left\langle \mathbf{I} - \left( \mathbf{I} - \frac{\eta}{2} \mathbf{H} \right)^N, \left( \mathbf{I} - \left( \mathbf{I} - \frac{\eta}{2} \mathbf{H} \right)^N \right) \cdot \mathbf{H}^{-1} \right\rangle.
\end{aligned}
$$

Noting that for any $x \in (0, 1/\eta)$, we have $1 - (1 - \eta x)^N \leq \min\{1, N\eta x\}$, then

$$\mathbf{I} - \left( \mathbf{I} - \frac{\eta}{2} \mathbf{H} \right)^N \preceq \mathbf{I}_{0:k} + N \frac{\eta}{2} \mathbf{H}_{k:\infty}.$$

Final results can be obtained:

variance error:

$$
\leq \frac{\sigma^2}{(1-\eta(\alpha\operatorname{tr}(\mathbf{H})))N}(k^* + \frac{N^2\eta^2}{4}\sum_{i>k^*}\lambda_i^2) + \frac{4\Gamma^2 f^2}{(1-\eta(\alpha\operatorname{tr}(\mathbf{H})))N}\operatorname{Tr}(\mathbf{H}_{0:k^*}^{-1} + \frac{N^2\eta^2}{4}\sum_{i>k^*}\lambda_i)
$$
(12)

$$
\leq \frac{\sigma^2}{(1-\eta(\alpha\operatorname{tr}(\mathbf{H})))N}(k^* + \frac{N^2\eta^2}{4}\sum_{i>k^*}\lambda_i^2) + \frac{4\Gamma^2 f^2}{N(1-\eta(\alpha\operatorname{tr}(\mathbf{H})))}(k^* N\eta + \frac{N^2\eta^2}{4}\sum_{i>k^*}\lambda_i)
$$
(13)

$$
\lesssim \frac{\sigma^2}{N}(k^* + N^2\eta^2\sum_{i>k^*}\lambda_i^2) + \frac{\Gamma^2 f^2}{N}(\sum_{i<k^*}\lambda_i^{-1} + N^2\eta^2\cdot\sum_{i>k^*}\lambda_i)
$$

$$
\lesssim \frac{\sigma^2}{N}(k^* + N^2\eta^2\sum_{i>k^*}\lambda_i^2) + \Gamma^2 f^2\eta(k^* + N\eta\sum_{i>k^*}\lambda_i)
$$

where the second inequality holds since $\operatorname{tr}(\mathbf{H}_{0:k^*}^{-1}) = \sum_{i=1}^{k^*}\lambda_i^{-1} \leq N\eta\cdot k^*$ and $\lambda_i \geq \frac{1}{\eta N}$.

**Lower bound for variance error**

Similarly, we derive a recursion for the variance term to establish the lower bound:

$$
\begin{aligned}
\mathbf{C}_t &= (\mathcal{I} - \eta\mathcal{T}(\eta))\circ\mathbf{C}_{t-1} + \eta^2\boldsymbol{\Sigma} \\
&= (\mathcal{I} - \eta\widetilde{\mathcal{T}}(\eta))\circ\mathbf{C}_{t-1} + \eta^2(\mathcal{M} - \frac{1}{4}\widetilde{\mathcal{M}})\circ\mathbf{C}_{t-1} + \eta^2\boldsymbol{\Sigma} \\
&\succeq (\mathcal{I} - \eta\widetilde{\mathcal{T}}(\eta))\circ\mathbf{C}_{t-1} + \eta^2(\sigma^2\mathbf{H} + 4\widetilde{\Gamma}^2 f^2\mathbf{I}).
\end{aligned}
$$

It implies that

$$
\begin{aligned}
\mathbf{C}_t &= \eta^2\cdot\sum_{k=0}^{t-1}(\mathbf{I} - \frac{\eta}{2}\mathbf{H})^k(\sigma^2\mathbf{H} + 4\widetilde{\Gamma}^2 f^2\mathbf{I})(\mathbf{I} - \frac{\eta}{2}\mathbf{H})^{2k} \\
&\succeq \eta^2\cdot\sum_{k=0}^{t-1}(\mathbf{I} - \frac{\eta}{2}\mathbf{H})^k(\sigma^2\mathbf{H} + 4\widetilde{\Gamma}^2 f^2\mathbf{I}) \\
&= \eta^2\sigma^2\cdot(\mathbf{I} - (\mathbf{I} - \frac{\eta}{2}\mathbf{H})^{2t})\cdot(\eta\mathbf{I} - \frac{\eta^2}{4}\mathbf{H})^{-1} + 4\eta^2\widetilde{\Gamma}^2 f^2\cdot(\mathbf{I} - (\mathbf{I} - \frac{\eta}{2}\mathbf{H})^{2t})\cdot(\eta\mathbf{H} - \frac{\eta^2}{4}\mathbf{H}\mathbf{H})^{-1} \\
&\succeq \eta\sigma^2\cdot(\mathbf{I} - (\mathbf{I} - \frac{\eta}{2}\mathbf{H})^{2t}) + 4\eta\widetilde{\Gamma}^2 f^2\cdot(\mathbf{I} - (\mathbf{I} - \frac{\eta}{2}\mathbf{H})^{2t})\cdot\mathbf{H}^{-1}.
\end{aligned}
$$

where the last inequality comes from $\eta\mathbf{I} - \frac{\eta^2}{4}\mathbf{H} \preceq \eta\mathbf{I}$.

Applying Lemma D.5, we could derive the lower bound for variance error:

$$\text{variance error} \geq \frac{1}{2\eta N^2} \sum_{t=0}^{N-1} \sum_{k=t}^{N-1} \langle (\mathbf{I} - \frac{\eta}{2}\mathbf{H})^{k-t}\mathbf{H}, \mathbf{C}_t \rangle$$

$$\geq \frac{\sigma^2}{N^2} \sum_{t=0}^{N-1} \langle \mathbf{I} - (\mathbf{I} - \frac{\eta}{2}\mathbf{H})^{N-t}, (\mathbf{I} - (\mathbf{I} - \frac{\eta}{2}\mathbf{H})^{2t}) \rangle$$

$$+ \frac{4\widetilde{\Gamma}^2 f^2}{N^2} \sum_{t=0}^{N-1} \langle \mathbf{I} - (\mathbf{I} - \frac{\eta}{2}\mathbf{H})^{N-t}, (\mathbf{I} - (\mathbf{I} - \frac{\eta}{2}\mathbf{H})^{2t}) \cdot \mathbf{H}^{-1} \rangle$$

$$\geq \frac{\sigma^2}{N^2} \sum_i \sum_{t=0}^{N-1} (1 - (1 - \frac{\eta}{2}\lambda_i)^{N-t})(1 - (1 - \frac{\eta}{2}\lambda_i)^{2t})$$

$$+ \frac{4\widetilde{\Gamma}^2 f^2}{N^2} \sum_i \sum_{t=0}^{N-1} (1 - (1 - \frac{\eta}{2}\lambda_i)^{N-t})(1 - (1 - \frac{\eta}{2}\lambda_i)^{2t})\lambda_i^{-1}$$

$$\geq \frac{\sigma^2}{N^2} \sum_i \sum_{t=0}^{N-1} (1 - (1 - \frac{\eta}{2}\lambda_i)^{N-t-1})(1 - (1 - \frac{\eta}{2}\lambda_i)^{t})$$

$$+ \frac{4\widetilde{\Gamma}^2 f^2}{N^2} \sum_i \sum_{t=0}^{N-1} (1 - (1 - \frac{\eta}{2}\lambda_i)^{N-t-1})(1 - (1 - \frac{\eta}{2}\lambda_i)^{t})\lambda_i^{-1}.$$

By defining $f(x) := \sum_{t=0}^{N-1}(1 - (1 - x)^{N-t-1})(1 - (1 - x)^t)$ and according to Lemma C.3 in [56], we have:

$$\text{variance error} \geq \frac{\sigma^2}{N^2} \sum_i f(\eta\lambda_i) + \frac{4\widetilde{\Gamma}^2 f^2}{N^2} \sum_i f(\eta\lambda_i)\lambda_i^{-1}$$

$$\geq \frac{\sigma^2}{N^2}(\frac{Nk^*}{10} + \frac{2N^3}{25}\eta^2 \cdot \sum_{i>k^*} \lambda_i^2) + \frac{4\widetilde{\Gamma}^2 f^2}{N^2}(\frac{N}{10} \sum_{i<k^*} \lambda_i^{-1} + \frac{2N^3}{25}\eta^2 \cdot \sum_{i>k^*} \lambda_i)$$

$$\tag{14}$$

$$\geq \frac{\sigma^2}{15N}(k^* + N^2\eta^2 \cdot \sum_{i>k^*} \lambda_i^2) + \frac{4\widetilde{\Gamma}^2 f^2}{15N}(\sum_{i<k^*} \lambda_i^{-1} + N^2\eta^2 \cdot \sum_{i>k^*} \lambda_i)$$

$$\gtrsim \frac{\sigma^2}{N}(k^* + N^2\eta^2 \cdot \sum_{i>k^*} \lambda_i^2) + \frac{\widetilde{\Gamma}^2 f^2}{N}(\sum_{i<k^*} \lambda_i^{-1} + N^2\eta^2 \cdot \sum_{i>k^*} \lambda_i).$$

**Upper and Lower bound for bias error**

According to Theorem B.5 in [49], the upper and lower bounds on the bias error can be directly derived:

**Lemma D.6.** *Suppose Assumption 3.6 holds and the step size satisfies $\eta \leq 1/\lambda_1$, then it holds that:*

$$\text{bias} \lesssim \frac{1}{\eta^2 N^2} \cdot \|\mathbf{w}_0 - \mathbf{w}_*\|^2_{\mathbf{H}_{0:k^*}^{-1}} + \|\mathbf{w}_0 - \mathbf{w}_*\|^2_{\mathbf{H}_{k^*:\infty}}$$

$$+ \frac{\alpha(\|\mathbf{w}_0 - \mathbf{w}_*\|^2_{\mathbf{I}_{0:k^*}} + N\eta\|\mathbf{w}_0 - \mathbf{w}_*\|^2_{\mathbf{H}_{k^*:\infty}})}{N\eta} \cdot (\frac{k^*}{N} + N\eta^2 \sum_{i>k^*} \lambda_i^2)$$

$$\text{bias} \gtrsim \frac{1}{\eta^2 N^2} \cdot \|\mathbf{w}_0 - \mathbf{w}_*\|^2_{\mathbf{H}_{0:k^*}^{-1}} + \|\mathbf{w}_0 - \mathbf{w}_*\|^2_{\mathbf{H}_{k^*:\infty}}$$

$$+ \frac{\beta(\|\mathbf{w}_0 - \mathbf{w}_*\|^2_{\mathbf{I}_{0:k^*}} + N\eta\|\mathbf{w}_0 - \mathbf{w}_*\|^2_{\mathbf{H}_{k^*:\infty}})}{N\eta} \cdot (\frac{k^*}{N} + N\eta^2 \sum_{i>k^*} \lambda_i^2)$$

By integrating the aforementioned results, we thus complete the proof.

# E DP-TAGLMtron

## E.1 Privacy Guarantee

**Lemma E.1** ([22]). *Algorithm 4 guarantees $(\tau, \frac{\tau\lceil\lg(N+1)\rceil}{2\sigma^2})$-Rényi differential privacy, where $N$ is the number of samples. Setting $\sigma = \frac{\sqrt{2\lceil\lg(N+1)\rceil\ln(1/\delta)}}{\varepsilon}$, one can guarantee $(\varepsilon, \delta)$-differential privacy, for $\varepsilon \leq 2\ln(1/\delta)$.*

To guarantee the privacy of Algorithm 3, we should first ensure that each step of DP-TAGLMtron is private.

We first consider $\{l_t\}_{t=0}^{N-1}$. According to Lemma E.1, it ensures that DP-tree-aggregation step is $(\varepsilon, \delta)$-DP with $f = \frac{\sqrt{2\lceil\lg(N+1)\rceil\ln(1/\delta)}}{\varepsilon}$.

Moreover, denoting $\varepsilon' = \varepsilon/\sqrt{8N\log(2/\delta)}$ and $\delta' = \delta/(2N)$, according to Lemma 2.3 in [24], we could know that $\{s_t\}_{t=0}^{N}$ is $(\varepsilon', \delta')$-DP.

Combining with the advanced composition theorem, it holds that Algorithm 3 is $(\varepsilon, \delta)$-DP.

## E.2 Utility Guarantee

Here we introduce several lemmas for future proof.

**Lemma E.2.** *Given a sequence of Positive Semi-Definite (PSD) matrices $\mathbf{P}_t, t = 0\ldots N$, where $\mathbf{P}_t = (\mathcal{I} - \frac{\eta}{2} \cdot \mathcal{T}(2\eta)) \circ \mathbf{P}_{t-1}$ and $\mathbf{P}_0 = \mathbf{I}$, if $\eta \leq 1/(4\beta\operatorname{tr}(\mathbf{H})\log N)$, then for any $t \in [0, N]$, the following inequality holds:*

$$\operatorname{tr}(\mathbf{H}\mathbf{P}_t) \leq 2 \cdot \langle \frac{1}{\eta N}\mathbf{I}_{0:k^*} + \mathbf{H}_{k^*:\infty}, \mathbf{I}\rangle,$$

*where $k^* \in [N]$ can be arbitrarily chosen.*

**Lemma E.3.** *Under Assumption 3.4(A), for every $\mathbf{A} \succeq \mathbf{0}$, it holds that*

$$(\mathcal{M} - \widetilde{\mathcal{M}}) \circ \mathcal{T}^{-1} \circ \mathbf{A} \preceq \mathcal{M} \circ \mathcal{T}^{-1} \circ \mathbf{A} \preceq \frac{2\alpha\operatorname{tr}(\mathbf{A})}{1 - \alpha\eta\operatorname{tr}(\mathbf{H})} \cdot \mathbf{H} \preceq 2\alpha\operatorname{tr}(\mathbf{A})\mathbf{H}.$$

**Lemma E.4** (Generic bounds on the GLM-tron iterates [49]). *Suppose that Assumption 3.6 holds. Consider the GLM-tron algorithm, we have the following recursion:*

- $\mathbf{A}_t \preceq \mathbf{A}_{t-1} - \frac{\eta}{2}(\mathbf{H}\mathbf{A}_{t-1} + \mathbf{A}_{t-1}\mathbf{H}) + \eta^2\mathcal{M} \circ \mathbf{A}_{t-1} + \eta^2\sigma^2\mathbf{H} := (\mathcal{I} - \frac{\eta}{2} \cdot \mathcal{T}(2\eta)) \circ \mathbf{A}_{t-1} + \eta^2\sigma^2\mathbf{H}$

- $\mathbf{A}_t \succeq \mathbf{A}_{t-1} - \frac{\eta}{2}(\mathbf{H}\mathbf{A}_{t-1} + \mathbf{A}_{t-1}\mathbf{H}) + \frac{\eta^2}{4}\mathcal{M} \circ \mathbf{A}_{t-1} + \eta^2\sigma^2\mathbf{H} := (\mathcal{I} - \frac{\eta}{2} \cdot \mathcal{T}(\frac{\eta}{2})) \circ \mathbf{A}_{t-1} + \eta^2\sigma^2\mathbf{H}$

*where $\mathbf{A}_t := \mathbb{E}(\mathbf{w}_t - \mathbf{w}_*)(\mathbf{w}_t - \mathbf{w}_*)^\top, t \geq 0$.*

**Lemma E.5** (Generic bounds on the DP-TAGLMtron iterates). *Suppose that Assumptions 3.6 (A) hold. Consider the DP-TAGLMtron algorithm, we have the following recursion:*

$$\mathbf{A}_t \preceq (\mathcal{I} - \frac{\eta}{2} \cdot \mathcal{T}(2\eta)) \circ \mathbf{A}_{t-1} + \eta^2\sigma^2\mathbf{H} + \eta^2 f^2(\lceil\log_2 N\rceil + 1)\mathbb{E}[\gamma^4(t) + \gamma^2(t)]\mathbf{I}.$$

$$+ \eta^2 f^2(\lceil\log_2 N\rceil + 1)(\sum_{j \in \mathcal{Q}(t)}(\mathcal{I} - \frac{\eta}{2} \cdot \widetilde{\mathcal{T}}(2\eta))^p \circ \mathbf{I} + 4\eta\alpha|\mathcal{Q}(t)|\operatorname{tr}(\mathbf{H})(\mathbf{I} - (\mathbf{I} - \frac{\eta}{2}\mathbf{H})^N))$$

$$\mathbf{A}_t \succeq \mathbf{A}_{t-1} - \frac{\eta}{2}(\mathbf{H}\mathbf{A}_{t-1} + \mathbf{A}_{t-1}\mathbf{H}) + \frac{\eta^2}{4}\mathcal{M} \circ \mathbf{A}_{t-1} + \eta^2\sigma^2\mathbf{H} + 4\eta^2\widetilde{\Gamma}^2 f^2\mathbf{I}$$

*where $\mathbf{A}_t := \mathbb{E}(\mathbf{w}_t - \mathbf{w}_*)(\mathbf{w}_t - \mathbf{w}_*)^\top, t \geq 0$.*

**Proof of Lemma E.5.** According to the update rule of DP-TAGLMtron, we know:

$$\mathbf{w}_{t+1} = \mathbf{w}_0 - \eta l'_{\leq t},$$

where $l'_{\leq t}$ is the private estimation of $\sum_{i=0}^{t} l_i$ generated by tree aggregation protocol. From [22], we notice $l'_{\leq t} = \sum_{i=0}^{t} l_i + \mathbf{v}_t$, where $l_i = (\mathbf{x}_t(\mathrm{ReLU}(\mathbf{x}_t^\top \mathbf{w}_t) - y_t))$ and $\mathbf{v}_t$ is a combination of at most $O(\lceil \log_2 t \rceil)$ Gaussian noise. If we set $\mathbf{w}_0 = 0$, then it holds that:

$$
\begin{aligned}
\mathbf{w}_{t+1} &= \mathbf{w}_0 - \eta \cdot \left( \sum_{i=0}^{t} l_i + \mathbf{v}_t \right) \\
&= \mathbf{w}_0 - \eta \cdot \sum_{i=0}^{t} (\mathbf{x}_t(\mathrm{ReLU}(\mathbf{x}_t^\top \mathbf{w}_t) - y_t)) - \eta \cdot \mathbf{v}_t \\
&= \mathbf{w}_t - \eta(\mathbf{x}_t(\mathrm{ReLU}(\mathbf{x}_t^\top \mathbf{w}_t) - y_t) + \mathbf{v}_t - \mathbf{v}_{t-1}).
\end{aligned}
$$

By denoting $\boldsymbol{\Gamma}_t = \mathbf{v}_t - \mathbf{v}_{t-1}$ and noticing that $\mathbf{v}_t$ is a combination of at most $O(\lceil \log_2 t \rceil)$ Gaussian noise, it implies that $\boldsymbol{\Gamma}_t$ is a combination of at most $\lceil \log_2 t + 1 \rceil$ random Gaussian vectors, i.e. $\boldsymbol{\Gamma}_t = \sum_{q \in \mathcal{Q}(t)} \gamma(q) \Gamma f^2 \mathbf{g}_q$, where $\mathcal{Q}(t)$ is an index set and $\mathcal{Q}(t) = \{2^{Q-1}, 2^{Q-1}+2^{Q-2}, \ldots, 2^{Q-1} + 2^{Q-2} + \cdots + 2^0\}$ when $t = 2^Q$ with $Q \in \mathbb{N}$ and $\gamma(q) = \max\{\gamma_1, \ldots, \gamma_q\}$. Moreover, we know $\mathbf{g}_q \sim \mathcal{N}(0, \mathbf{I}_{d \times d})$, indicating:

$$
\begin{aligned}
\mathbf{w}_t - \mathbf{w}_* &= (\mathbf{I} - \eta \mathbb{1}[\mathbf{x}_t^\top \mathbf{w}_{t-1} > 0] \mathbf{x}_t \mathbf{x}_t^\top)(\mathbf{w}_{t-1} - \mathbf{w}_*) \\
&\quad + \eta(\mathbb{1}[\mathbf{x}_t^\top \mathbf{w}_* > 0] - \mathbb{1}[\mathbf{x}_t^\top \mathbf{w}_{t-1} > 0]) \mathbf{x}_t \mathbf{x}_t^\top \mathbf{w}_* + \eta z_t \mathbf{x}_t - \eta \boldsymbol{\Gamma}_t.
\end{aligned}
$$

Let us consider the expected outer product:

$$
\begin{aligned}
\mathbf{A}_t &= \mathbb{E}[\|\mathbf{w}_t - \mathbf{w}_*\|_2^2] = \mathbb{E}[(\mathbf{w}_t - \mathbf{w}_*)(\mathbf{w}_t - \mathbf{w}_*)^\top] \\
&= \mathbb{E} \underbrace{\left( (\mathbf{I} - \eta \mathbb{1}[\mathbf{x}_t^\top \mathbf{w}_{t-1} > 0] \cdot \mathbf{x}_t \mathbf{x}_t^\top) \cdot (\mathbf{w}_{t-1} - \mathbf{w}_*))((\mathbf{I} - \eta \mathbb{1}[\mathbf{x}_t^\top \mathbf{w}_{t-1} > 0] \cdot \mathbf{x}_t \mathbf{x}_t^\top) \cdot (\mathbf{w}_{t-1} - \mathbf{w}_*)) \right)^\top}_{\text{(quadratic term 1)}} \\
&\quad + \eta^2 \cdot \mathbb{E} \underbrace{\left( \mathbb{1}[\mathbf{x}_t^\top \mathbf{w}_* > 0] - \mathbb{1}[\mathbf{x}_t^\top \mathbf{w}_{t-1} > 0])^2 \cdot (\mathbf{x}_t \mathbf{x}_t^\top \mathbf{w}_*)(\mathbf{x}_t \mathbf{x}_t^\top \mathbf{w}_*) \right)^\top}_{\text{(quadratic term 2)}} \\
&\quad + 2\eta \cdot \mathbb{E} \underbrace{(\mathbb{1}[\mathbf{x}_t^\top \mathbf{w}_* > 0] - \mathbb{1}[\mathbf{x}_t^\top \mathbf{w}_{t-1} > 0]) \cdot (\mathbf{I} - \eta \mathbb{1}[\mathbf{x}_t^\top \mathbf{w}_{t-1} > 0] \cdot \mathbf{x}_t \mathbf{x}_t^\top)(\mathbf{w}_{t-1} - \mathbf{w}_*)(\mathbf{x}_t \mathbf{x}_t^\top \mathbf{w}_*)^\top}_{\text{(crossing term 1)}} \\
&\quad - \eta \cdot \underbrace{(\mathbb{E}[(\mathbf{I} - \eta \mathbb{1}[\mathbf{x}_t^\top \mathbf{w}_{t-1} > 0] \mathbf{x}_t \mathbf{x}_t^\top)(\mathbf{w}_t - \mathbf{w}_*) \boldsymbol{\Gamma}_t^\top] + \mathbb{E}[\boldsymbol{\Gamma}_t (\mathbf{w}_t - \mathbf{w}_*)^\top (\mathbf{I} - \mathbb{1}[\mathbf{x}_t^\top \mathbf{w}_{t-1} > 0] \eta \mathbf{x}_t \mathbf{x}_t^\top)])}_{\text{(crossing term 2)}} \\
&\quad - \eta \cdot \underbrace{(\mathbb{E}[(\mathbb{1}[\mathbf{x}_t^\top \mathbf{w}_* > 0] - \mathbb{1}[\mathbf{x}_t^\top \mathbf{w}_{t-1} > 0])(\mathbf{x}_t \mathbf{x}_t^\top \mathbf{w}_*) \boldsymbol{\Gamma}_t^\top] + \mathbb{E}[\boldsymbol{\Gamma}_t (\mathbf{x}_t \mathbf{x}_t^\top \mathbf{w}_*)^\top (\mathbb{1}[\mathbf{x}_t^\top \mathbf{w}_* > 0] - \mathbb{1}[\mathbf{x}_t^\top \mathbf{w}_{t-1} > 0])])}_{\text{(crossing term 3)}} \\
&\quad + \eta^2 \cdot \underbrace{\mathbb{E}(\sigma^2 \mathbf{x}_t^\top \mathbf{x}_t)}_{\text{(quadratic term 3)}} + \eta^2 \cdot \underbrace{\mathbb{E}[\boldsymbol{\Gamma}_t \boldsymbol{\Gamma}_t^\top]}_{\text{(quadratic term 4)}}.
\end{aligned}
$$

It can be observed that quadratic terms 1,2,3 and crossing term 1 are original terms for non-private problems, where the remaining are introduced by the private noise.

According to Assumption 3.6, Lemma C.2 and the independence of $\mathbf{w}_*$, we have:

(crossing term 3) $= 0$,

(crossing term 2) $= \mathbb{E}[(\mathbf{I} - \eta \mathbb{1}[\mathbf{x}_t^\top \mathbf{w}_{t-1} > 0] \mathbf{x}_t \mathbf{x}_t^\top) \mathbf{w}_t \boldsymbol{\Gamma}_t^\top] + \mathbb{E}[\boldsymbol{\Gamma}_t \mathbf{w}_t^\top (\mathbf{I} - \mathbb{1}[\mathbf{x}_t^\top \mathbf{w}_{t-1} > 0] \eta \mathbf{x}_t \mathbf{x}_t^\top)]$.

Now we examine the crossing term 2, denoting $\mathbf{R}_t := (\mathbf{I} - \eta \mathbb{1}[\mathbf{x}_t^\top \mathbf{w}_{t-1} > 0] \mathbf{x}_t \mathbf{x}_t^\top)$. According to the update rule, we have:

$$
\begin{aligned}
\mathbf{w}_t &= \mathbf{w}_{t-1} - \eta \cdot (\mathrm{ReLU}(\mathbf{x}_t^\top \mathbf{w}_{t-1}) - y_t) \mathbf{x}_t + \boldsymbol{\Gamma}_t \\
&= (\mathbf{I} - \eta \mathbb{1}[\mathbf{x}_t^\top \mathbf{w}_{t-1} > 0] \cdot \mathbf{x}_t \mathbf{x}_t^\top) \mathbf{w}_{t-1} + \eta \mathbf{x}_t y_t + \boldsymbol{\Gamma}_t \\
&= \sum_{j=1}^{t-1} (\mathbf{R}_{t-1} \mathbf{R}_{t-2} \cdots \mathbf{R}_j (\eta \mathbf{x}_{j-1} y_{j-1} + \eta \boldsymbol{\Gamma}_{j-1})) + \eta \mathbf{x}_{t-1} y_{t-1} + \eta \boldsymbol{\Gamma}_{t-1}.
\end{aligned}
$$

Hence, we have:

$$\mathbf{w}_t \boldsymbol{\Gamma}_t^\top = \sum_{j=1}^{t-1} (\mathbf{R}_{t-1}\mathbf{R}_{t-2}\cdots\mathbf{R}_j(\eta\mathbf{x}_{j-1}y_{j-1} + \eta\boldsymbol{\Gamma}_{j-1}))\boldsymbol{\Gamma}_t^\top + \eta\mathbf{x}_{t-1}\boldsymbol{\Gamma}_t^\top y_{t-1} + \eta\boldsymbol{\Gamma}_{t-1}\boldsymbol{\Gamma}_t^\top.$$

By multiplying both sides by A and taking the expected value, we can obtain:

$$\mathbb{E}[(\mathbf{I} - \eta\mathbf{x}_t\mathbf{x}_t^\top)\mathbf{w}_t\boldsymbol{\Gamma}_t^\top] = \mathbb{E}[\eta\mathbf{R}_t\sum_{j=1}^{t-1}(\mathbf{R}_{t-1}\mathbf{R}_{t-2}\cdots\mathbf{R}_j\boldsymbol{\Gamma}_{j-1})\boldsymbol{\Gamma}_t^\top] + \mathbb{E}[\eta\mathbf{R}_t\mathbf{x}_{t-1}y_{t-1}\boldsymbol{\Gamma}_t^\top] + \mathbb{E}[\eta\mathbf{R}_t\boldsymbol{\Gamma}_{t-1}\boldsymbol{\Gamma}_t^\top].$$

We first consider the second term, notice that:

$$\mathbb{E}[\eta\mathbf{R}_t\mathbf{x}_{t-1}y_{t-1}\boldsymbol{\Gamma}_t^\top] = \mathbb{E}[\eta\mathbf{R}_t(\mathbb{1}[\mathbf{x}_t^\top\mathbf{w}_* > 0]\cdot\mathbf{x}_t\mathbf{x}_t^\top\mathbf{w}_* + z_t\mathbf{x}_t)\boldsymbol{\Gamma}_t^\top] = 0.$$

Furthermore, considering $\boldsymbol{\Gamma}_t$ and $\boldsymbol{\Gamma}_j$, when $j < t$ and $j \notin \mathcal{Q}(t)$, it follows that $\mathbb{E}[\boldsymbol{\Gamma}_j\boldsymbol{\Gamma}_t^\top] = 0$, otherwise, we have:

$$\mathbb{E}[\boldsymbol{\Gamma}_j\boldsymbol{\Gamma}_t^\top] = (\sum_{q\in\mathcal{Q}(j-1)}\gamma(q)f^2\mathbf{g}_q)(\sum_{q'\in\mathcal{Q}(t)}\gamma(q')f^2\mathbf{g}_{q'})^\top.$$

These results arise because only $\boldsymbol{\Gamma}_q$ is dependent on $\mathbf{g}_q$ for $q \in \mathcal{Q}(t)$. Hence, denoting $\mathbf{R}_{[t:j]} = \mathbf{R}_t\cdots\mathbf{R}_j$, we have:

$$\mathbb{E}[\eta\mathbf{R}_t\sum_{j=1}^{t-1}(\mathbf{R}_{t-1}\mathbf{R}_{t-2}\cdots\mathbf{R}_j\boldsymbol{\Gamma}_{j-1})\boldsymbol{\Gamma}_t^\top]$$

$$= \eta\mathbb{E}[\mathbf{R}_t\sum_{j\in\mathcal{Q}(t)}(\mathbf{R}_{t-1}\mathbf{R}_{t-2}\cdots\mathbf{R}_j\boldsymbol{\Gamma}_{j-1})\boldsymbol{\Gamma}_t^\top]$$

$$= \eta\mathbb{E}[\sum_{j\in\mathcal{Q}(t)}\mathbf{R}_{[t:j]}(\sum_{q\in\mathcal{Q}(j-1)}\gamma(q)f^2\mathbf{g}_q)(\sum_{q'\in\mathcal{Q}(t)}\gamma(q')f^2\mathbf{g}_{q'})^\top]$$

$$= \eta\mathbb{E}[\sum_{j\in\mathcal{Q}(t)}\mathbf{R}_{[t:j]}(\sum_{\tilde{q}\in\mathcal{Q}(j-1)\cap\mathcal{Q}(t)}\gamma^2(\tilde{q})f^2\mathbf{g}_{\tilde{q}}\mathbf{g}_{\tilde{q}}^\top + \sum_{q\neq q'}\gamma(q')\gamma(q)f^2\mathbf{g}_q\mathbf{g}_{q'}^\top)]$$

$$= \eta f^2\sum_{j\in\mathcal{Q}(t)}\sum_{\tilde{q}\in\mathcal{Q}(j-1)\cap\mathcal{Q}(t)}\mathbb{E}[\gamma^2(\tilde{q})f^2\mathbf{R}_{[t:j]}].$$

For the last term of crossing term 3, we could also derive:

$$\mathbb{E}[\eta\mathbf{R}_t\boldsymbol{\Gamma}_{t-1}\boldsymbol{\Gamma}_t^\top] = \sum_{\tilde{q}\in\mathcal{Q}(t-1)\cap\mathcal{Q}(t)}\mathbb{E}[\gamma^2(\tilde{q})f^2\mathbf{R}_t].$$

Combining the above results with the symmetric of $\mathbf{R}_t$, it implies that:

$$-(\text{crossing term 2}) = -\eta f^2\sum_{j\in\mathcal{Q}(t)}\sum_{\tilde{q}\in\mathcal{Q}(j-1)\cap\mathcal{Q}(t)}\mathbb{E}[\gamma^2(\tilde{q})\mathbf{R}_{[t:j]} + \gamma^2(\tilde{q})\mathbf{R}_{[j:t]}]$$

$$\preceq \eta f^2(\lceil\log_2 N\rceil + 1)\sum_{j\in\mathcal{Q}(t)}\mathbb{E}[\mathbf{R}_t\ldots\mathbf{R}_j\mathbf{R}_j\ldots\mathbf{R}_t] + \eta^2 f^2(\lceil\log_2 N\rceil + 1)\mathbb{E}[\gamma^4(t)]\mathbf{I}.$$

Recall $\mathbf{R}_t := (\mathbf{I} - \eta\mathbb{1}[\mathbf{x}_t^\top\mathbf{w}_{t-1} > 0]\mathbf{x}_t\mathbf{x}_t^\top)$ and the lemma in [49].

It indicates that:

$$\mathbb{E}[\mathbf{R}_t\ldots\mathbf{R}_j\mathbf{R}_j\ldots\mathbf{R}_t] \preceq (\mathcal{I} - \frac{\eta}{2}\cdot\mathcal{T}(2\eta))^{t-j+1}\circ\mathbf{I}.$$

We first consider $\mathbf{P}_p = (\mathcal{I} - \frac{\eta}{2} \cdot \mathcal{T}(2\eta)) \circ \mathbf{P}_{p-1}$ and $\mathbf{P}_0 = \mathbf{I}$. Then, it follows that:

$$
\begin{aligned}
\mathbf{P}_p &\preceq (\mathcal{I} - \frac{\eta}{2} \cdot \widetilde{\mathcal{T}}(2\eta)) \circ \mathbf{P}_{p-1} + \eta^2 (\mathcal{M} - \widetilde{\mathcal{M}}) \circ \mathbf{P}_{p-1} \\
&\preceq (\mathcal{I} - \frac{\eta}{2} \cdot \widetilde{\mathcal{T}}(2\eta)) \circ \mathbf{P}_{p-1} + \eta^2 \alpha \operatorname{tr}(\mathbf{H}\mathbf{P}_{p-1})\mathbf{H} \\
&= (\mathcal{I} - \frac{\eta}{2} \cdot \widetilde{\mathcal{T}}(2\eta))^p \circ \mathbf{I} + 2\eta^2 \alpha \operatorname{tr}(\mathbf{H}) \sum_{i=0}^{p} (\mathcal{I} - \frac{\eta}{2} \cdot \widetilde{\mathcal{T}}(2\eta))^i \mathbf{H} \\
&\preceq (\mathcal{I} - \frac{\eta}{2} \cdot \widetilde{\mathcal{T}}(2\eta))^p \circ \mathbf{I} + 4\eta\alpha \operatorname{tr}(\mathbf{H})(\mathbf{I} - (\mathbf{I} - \frac{\eta}{2}\mathbf{H})^N),
\end{aligned}
$$

where the term $\mathbf{H}\mathbf{P}_{p-1}$ can be bounded by Lemma E.2. Now, we could derive:

$$
\sum_{j \in \mathcal{Q}(t)} \mathbb{E}[\mathbf{R}_t \ldots \mathbf{R}_j \mathbf{R}_j \ldots \mathbf{R}_t] \preceq \sum_{j \in \mathcal{Q}(t)} (\mathcal{I} - \frac{\eta}{2} \cdot \widetilde{\mathcal{T}}(2\eta))^p \circ \mathbf{I} + 4\eta\alpha|\mathcal{Q}(t)| \operatorname{tr}(\mathbf{H})(\mathbf{I} - (\mathbf{I} - \frac{\eta}{2}\mathbf{H})^N).
$$

As a results, the generic bounds on the DP-TAGLMtron iterate could have the following recursion:

$$
\begin{aligned}
\mathbf{A}_t &\preceq (\mathcal{I} - \frac{\eta}{2} \cdot \mathcal{T}(2\eta)) \circ \mathbf{A}_{t-1} + \eta^2 \sigma^2 \mathbf{H} + \eta^2 f^2 (\lceil \log_2 N \rceil + 1)\mathbb{E}[\gamma^4(t) + \gamma^2(t)]\mathbf{I}. \\
&+ \eta^2 f^2 (\lceil \log_2 N \rceil + 1)(\sum_{j \in \mathcal{Q}(t)} (\mathcal{I} - \frac{\eta}{2} \cdot \widetilde{\mathcal{T}}(2\eta))^p \circ \mathbf{I} + 4\eta\alpha|\mathcal{Q}(t)| \operatorname{tr}(\mathbf{H})(\mathbf{I} - (\mathbf{I} - \frac{\eta}{2}\mathbf{H})^N)).
\end{aligned}
$$

We now move on to the second part. In light of the $\mathbf{\Gamma}_t$ definition, we know it is at least a random Gaussian vector such that $\mathbf{\Gamma}_t = \widetilde{\Gamma}\mathbf{g}_t$, where $\widetilde{\Gamma} = \min \boldsymbol{\gamma}$. Moreover, it holds that

$$
\mathbb{E}[(\mathbf{I} - \eta\mathbf{x}_t\mathbf{x}_t^\top)\mathbf{w}_t\mathbf{\Gamma}_t^\top] = \mathbb{E}[\eta\mathbf{R}_t \sum_{j=1}^{t-1}(\mathbf{R}_{t-1}\mathbf{R}_{t-2} \cdots \mathbf{R}_j\mathbf{\Gamma}_{j-1})\mathbf{\Gamma}_t^\top] + \mathbb{E}[\eta\mathbf{R}_t\mathbf{x}_{t-1}y_{t-1}\mathbf{\Gamma}_t^\top] + \mathbb{E}[\eta\mathbf{R}_t\mathbf{\Gamma}_{t-1}\mathbf{\Gamma}_t^\top] = 0.
$$

Since $\mathbb{E}[\mathbf{\Gamma}_j\mathbf{\Gamma}_t^\top] = 0$ for any $j \neq t$. Therefore, the iteration of $\mathbf{A}_t$ will be the same case in DP-GLMtron. $\qquad \square$

Consequently, analogous to the previous case, we can decompose $\mathbf{A}_t$ into bias term $\mathbf{B}_t$ and variance term $\mathbf{C}_t$.

$$
\begin{aligned}
\mathbf{B}_t &= (\mathcal{I} - \frac{\eta}{2} \cdot \mathcal{T}(2\eta)) \circ \mathbf{B}_{t-1}, \quad \mathbf{B}_0 = (\mathbf{w}_0 - \mathbf{w}^*)(\mathbf{w}_0 - \mathbf{w}^*)^\top; \\
\mathbf{C}_t &= (\mathcal{I} - \frac{\eta}{2} \cdot \mathcal{T}(2\eta)) \circ \mathbf{C}_{t-1} + \eta^2 \mathbf{M}_t, \quad \mathbf{C}_0 = \mathbf{0}
\end{aligned}
$$

where

$$
\begin{aligned}
\mathbf{M}_t &\preceq \sigma^2 \mathbf{H} + f^2(\lceil \log_2 N \rceil + 1)\mathbb{E}[\gamma^4(t) + \gamma^2(t)]\mathbf{I} \\
&+ f^2(\lceil \log_2 N \rceil + 1)(\sum_{j \in \mathcal{Q}(t)} (\mathcal{I} - \frac{\eta}{2} \cdot \widetilde{\mathcal{T}}(2\eta))^p \circ \mathbf{I} + 4\eta\alpha|\mathcal{Q}(t)| \operatorname{tr}(\mathbf{H})(\mathbf{I} - (\mathbf{I} - \frac{\eta}{2}\mathbf{H})^N)).
\end{aligned}
$$

**Lemma E.6.** *Let* $\mathbf{M}$ *represent a union upper bound for* $\mathbf{M}_0, \ldots, \mathbf{M}_{N-1}$. *Under Assumptions (A) and 3.6, and provided that the step size* $\eta$ *satisfies* $\eta \lesssim 1/(\operatorname{tr}(\mathbf{H})\log N)$, *the following holds:*

$$
\mathbf{C}_0 \preceq \cdots \preceq \mathbf{C}_N \preceq \frac{8\alpha\eta^2}{1 - \alpha\eta \operatorname{tr}(\mathbf{H})} \cdot \langle \mathbf{I} - (\mathbf{I} - \frac{\eta}{2}\mathbf{H})^N, \mathbf{M}\rangle \cdot (\mathbf{I} - (\mathbf{I} - \frac{\eta}{2}\mathbf{H})^N) + \eta^2 \cdot \sum_{t=0}^{N-1}(\mathcal{I} - \frac{\eta}{2} \cdot \widetilde{\mathcal{T}}(2\eta))^t \circ \mathbf{M}.
\tag{15}
$$

**Proof of Lemma E.6.** Our proofs follow the main idea of [48], we include them here for completeness. According to the update rule of $\mathbf{C}_t$, it follows that:

$$
\mathbf{C}_t = \eta^2 \sum_{t=0}^{N-1}(\mathcal{I} - \frac{\eta}{2} \cdot \mathcal{T}(2\eta)) \circ \mathbf{M}_t.
$$

It can be observed that $\mathbf{C}_0 \preceq \mathbf{C}_1 \preceq \cdots \preceq \mathbf{C}_N$ since $(\mathcal{I} - \frac{\eta}{2} \cdot \mathcal{T}(2\eta))$ is a PSD mapping.

Let $\mathbf{M}$ be an union upper bound of $\{\mathbf{M}_0, \ldots, \mathbf{M}_{N-1}\}$, we will have:

$$\mathbf{C}_N \preceq \eta^2 \sum_{t=0}^{N-1} (\mathcal{I} - \frac{\eta}{2} \cdot \mathcal{T}(2\eta)) \circ \mathbf{M} = 2\eta \cdot \mathcal{T}^{-1} \circ (\mathcal{I} - (\mathcal{I} - \frac{\eta}{2} \cdot \mathcal{T}(2\eta))^N) \circ \mathbf{M} \preceq 2\eta \cdot \mathcal{T}^{-1} \circ (\mathcal{I} - (\mathcal{I} - \frac{\eta}{2} \cdot \widetilde{\mathcal{T}}(2\eta))^N) \circ \mathbf{M}.$$

The last inequality comes from $(\widetilde{\mathcal{T}} - \mathcal{T}), (\mathcal{I} - \frac{\eta}{2} \cdot \mathcal{T}(2\eta))$ and $(\mathcal{I} - \frac{\eta}{2} \cdot \widetilde{\mathcal{T}}(2\eta))$ are PSD mappings. Hence, it implies that:

$$\mathbf{C}_{t+1} = (\mathcal{I} - \frac{\eta}{2} \cdot \mathcal{T}(2\eta)) \circ \mathbf{C}_t + \eta^2 \cdot (\mathcal{M} - \widetilde{\mathcal{M}}) \circ \mathbf{C}_t + \eta^2 \mathbf{M}_t$$

$$\preceq (\mathcal{I} - \frac{\eta}{2} \cdot \mathcal{T}(2\eta)) \circ \mathbf{C}_t + \eta^2 \cdot (\mathcal{M} - \widetilde{\mathcal{M}}) \circ \mathbf{C}_N + \eta^2 \mathbf{M}$$

$$\preceq (\mathcal{I} - \frac{\eta}{2} \cdot \mathcal{T}(2\eta)) \circ \mathbf{C}_t + \eta^2 \cdot (\mathcal{M} - \widetilde{\mathcal{M}}) \circ 2\eta \cdot \mathcal{T}^{-1} \circ (\mathcal{I} - (\mathcal{I} - \frac{\eta}{2} \cdot \widetilde{\mathcal{T}}(2\eta))^N) \circ \mathbf{M} + \eta^2 \mathbf{M}$$

$$\preceq (\mathcal{I} - \frac{\eta}{2} \cdot \mathcal{T}(2\eta)) \circ \mathbf{C}_t + \frac{4\alpha\eta^3}{1 - \alpha\eta \operatorname{tr}(\mathbf{H})} \cdot \langle \mathbf{I} - (\mathbf{I} - \frac{\eta}{2}\mathbf{H})^N, \mathbf{M} \rangle \cdot \mathbf{H} + \eta^2 \mathbf{M},$$

where the last inequality comes from Lemma E.3. Therefore, we have:

$$\mathbf{C}_N \preceq \frac{4\alpha\eta^3}{1 - \alpha\eta \operatorname{tr}(\mathbf{H})} \cdot \langle \mathbf{I} - (\mathbf{I} - \frac{\eta}{2}\mathbf{H})^N, \mathbf{M} \rangle \cdot \sum_{t=0}^{N-1} (\mathcal{I} - \frac{\eta}{2} \cdot \widetilde{\mathcal{T}}(2\eta))^{N-1-t} \circ \mathbf{H} + \eta^2 \cdot \sum_{t=0}^{N-1} (\mathcal{I} - \frac{\eta}{2} \cdot \widetilde{\mathcal{T}}(2\eta))^{N-1-t} \circ \mathbf{M}$$

$$= \frac{8\alpha\eta^2}{1 - \alpha\eta \operatorname{tr}(\mathbf{H})} \cdot \langle \mathbf{I} - (\mathbf{I} - \frac{\eta}{2}\mathbf{H})^N, \mathbf{M} \rangle \cdot (\mathbf{I} - (\mathbf{I} - \frac{\eta}{2}\mathbf{H})^N) + \eta^2 \cdot \sum_{t=0}^{N-1} (\mathcal{I} - \frac{\eta}{2} \cdot \widetilde{\mathcal{T}}(2\eta))^t \circ \mathbf{M}.$$

$\square$

**Lemma E.7.** *Variance error] If the step size satisfies $\eta \lesssim 1/(\operatorname{tr}(\mathbf{H}) \log N)$, then the following holds for any $k \geq 1$,*

$$\text{variance error} \lesssim \frac{\alpha\eta}{N(1 - \alpha\eta \operatorname{tr}(\mathbf{H}))} \cdot \langle \mathbf{I}_{0:k^*} + \frac{N\eta}{2}\mathbf{H}_{k^*:\infty}, \widetilde{\mathbf{M}} \rangle \cdot (k^* + \frac{N^2\eta^2}{4} \sum_{i>k^*} \lambda_i^2) + \frac{1}{N} \cdot \langle \mathbf{H}_{0:k^*}^{-1} + \frac{N^2\eta^2}{4}\mathbf{H}_{k^*:\infty}, \widetilde{\mathbf{M}} \rangle,$$

*where*

$$\widetilde{\mathbf{M}} = \sigma^2 \cdot \mathbf{H} + \eta\alpha f^2 (\log_2 N)^2 \operatorname{tr}(\mathbf{H}) \cdot (\mathbf{I}_{0:k^*} + \frac{N\eta}{2}\mathbf{H}_{k^*:\infty}) + f^2 \log_2 N \cdot (\mathbb{E}[\gamma(N)^2 + \gamma(N)^4] + 1) \cdot \mathbf{I}.$$

*Proof.* According to Lemma D.5, we have:

$$\text{variance error} \leq \frac{16\alpha\eta}{N(1 - \alpha\eta \operatorname{tr}(\mathbf{H}))} \langle \mathbf{I} - (\mathbf{I} - \frac{\eta}{2}\mathbf{H})^N, \mathbf{C}_N \rangle$$

$$\leq \frac{16\alpha\eta}{N(1 - \alpha\eta \operatorname{tr}(\mathbf{H}))} \cdot \langle \mathbf{I} - (\mathbf{I} - \frac{\eta}{2}\mathbf{H})^N, \mathbf{M} \rangle \cdot \langle \mathbf{I} - (\mathbf{I} - \frac{\eta}{2}\mathbf{H})^N, \mathbf{I} - (\mathbf{I} - \frac{\eta}{2}\mathbf{H})^N \rangle$$

$$+ \frac{\eta}{N} \cdot \langle \mathbf{I} - (\mathbf{I} - \frac{\eta}{2}\mathbf{H})^N, \sum_{t=0}^{N-1} (\mathcal{I} - \frac{\eta}{2} \cdot \widetilde{\mathcal{T}}(2\eta))^t \circ \mathbf{M} \rangle$$

$$= \frac{16\alpha\eta}{N(1 - \alpha\eta \operatorname{tr}(\mathbf{H}))} \cdot \langle \mathbf{I} - (\mathbf{I} - \frac{\eta}{2}\mathbf{H})^N, \mathbf{M} \rangle \cdot \langle \mathbf{I} - (\mathbf{I} - \frac{\eta}{2}\mathbf{H})^N, \mathbf{I} - (\mathbf{I} - \frac{\eta}{2}\mathbf{H})^N \rangle$$

$$+ \frac{\eta}{N} \cdot \langle \sum_{t=0}^{N-1} (\mathcal{I} - \frac{\eta}{2} \cdot \widetilde{\mathcal{T}}(2\eta))^t \circ (\mathbf{I} - (\mathbf{I} - \frac{\eta}{2}\mathbf{H})^N), \mathbf{M} \rangle$$

$$= \frac{16\alpha\eta}{N(1 - \alpha\eta \operatorname{tr}(\mathbf{H}))} \cdot \langle \mathbf{I} - (\mathbf{I} - \frac{\eta}{2}\mathbf{H})^N, \mathbf{M} \rangle \cdot \langle \mathbf{I} - (\mathbf{I} - \frac{\eta}{2}\mathbf{H})^N, \mathbf{I} - (\mathbf{I} - \frac{\eta}{2}\mathbf{H})^N \rangle$$

$$+ \frac{2}{N} \cdot \langle (\mathbf{I} - (\mathbf{I} - \frac{\eta}{2}\mathbf{H})^N)^2 \mathbf{H}^{-1}, \mathbf{M} \rangle.$$

Furthermore, Lemma E.5 implies that, for any $t \leq N$:

$$\mathbf{M}_t \preceq \sigma^2 \mathbf{H} + f^2(\lceil \log_2 N \rceil + 1)\mathbb{E}[\gamma^4(t) + \gamma^2(t)]\mathbf{I}$$
$$+ f^2(\lceil \log_2 N \rceil + 1)(\sum_{j \in \mathcal{Q}(t)}(\mathcal{I} - \frac{\eta}{2} \cdot \widetilde{\mathcal{T}}(2\eta))^p \circ \mathbf{I} + 4\eta\alpha|\mathcal{Q}(t)|\operatorname{tr}(\mathbf{H})(\mathbf{I} - (\mathbf{I} - \frac{\eta}{2}\mathbf{H})^N)).$$

Now we consider an arbitrary PSD mapping $\mathbf{A}$ commuting with $\mathbf{H}$, it holds that:

$$\langle \mathbf{A}, \mathbf{M} \rangle \lesssim \sigma^2 \langle \mathbf{H}, \mathbf{A} \rangle + f^2 \log_2 N \cdot \mathbb{E}[\gamma^4(t) + \gamma^2(t)]\operatorname{tr}(\mathbf{A}) + \eta\alpha f^2(\log_2 N)^2 \operatorname{tr}(\mathbf{H}) \cdot \langle \mathbf{A}, \mathbf{I} - (\mathbf{I} - \frac{\eta}{2}\mathbf{H})^N \rangle.$$

Moreover, noting that for any $x \in (0, 2/\eta)$, we have $1 - (1 - \frac{\eta x}{2})^N \leq \min\{1, N\eta x/2\}$, then

$$\mathbf{I} - (\mathbf{I} - \frac{\eta}{2}\mathbf{H})^N \preceq \mathbf{I}_{0:k^*} + \frac{N\eta}{2}\mathbf{H}_{k^*:\infty},$$

for any $k^* \geq 0$. Therefore, we can define a new matrix as follows:

$$\widetilde{\mathbf{M}} = \sigma^2 \cdot \mathbf{H} + \eta\alpha f^2(\log_2 N)^2 \operatorname{tr}(\mathbf{H}) \cdot (\mathbf{I}_{0:k^*} + \frac{N\eta}{2}\mathbf{H}_{k^*:\infty}) + f^2 \log_2 N \cdot (\mathbb{E}[\gamma(N)^2 + \gamma(N)^4] + 1) \cdot \mathbf{I}.$$

Combining the previous results, we obtain

variance

$$\lesssim \frac{\alpha\eta}{N(1 - \alpha\eta\operatorname{tr}(\mathbf{H}))} \cdot \langle \mathbf{I}_{0:k^*} + \frac{N\eta}{2}\mathbf{H}_{k^*:\infty}, \widetilde{\mathbf{M}} \rangle \cdot (k^* + \frac{N^2\eta^2}{4}\sum_{i>k^*}\lambda_i^2) + \frac{1}{N} \cdot \langle \mathbf{H}_{0:k^*}^{-1} + \frac{N^2\eta^2}{4}\mathbf{H}_{k^*:\infty}, \widetilde{\mathbf{M}} \rangle$$

$$\lesssim \frac{\sigma^2}{N}(1 + \alpha\eta(\sum_{i<k^*}\lambda_i + \frac{N\eta}{2}\sum_{i>k^*}\lambda_i^2)) \cdot (k^* + \frac{N^2\eta^2}{4}\sum_{i>k^*}\lambda_i^2)$$

$$+ \frac{f^2 \log_2 N}{N} \cdot (\alpha\eta \log_2 N \operatorname{tr}(\mathbf{H}) + \mathbb{E}[\gamma(N)^2 + \gamma(N)^4]) \cdot (\alpha\eta(k^* + \frac{N\eta}{2}\sum_{i>k^*}\lambda_i) + (\sum_{i<k^*}\lambda_i^{-1} + \frac{N^2\eta^2}{4}\sum_{i>k^*}\lambda_i)).$$

$$\square$$

Now, we introduce the lemma in [49] to complete the proof.

**Lemma E.8** ([49]). *If the step size satisfies $\eta \leq 1/\lambda_1$, then it holds that for any $k^* \geq 1$,*

$$\text{bias} \lesssim \frac{1}{\eta^2 N^2} \cdot \|\mathbf{w}_0 - \mathbf{w}^*\|_{\mathbf{H}_{0:k^*}^{-1}}^2 + \|\mathbf{w}_0 - \mathbf{w}^*\|_{\mathbf{H}_{k^*:\infty}}^2$$

$$+ \frac{\alpha(\|\mathbf{w}_0 - \mathbf{w}^*\|_{\mathbf{I}_{0:k^*}}^2 + N\eta\|\mathbf{w}_0 - \mathbf{w}^*\|_{\mathbf{H}_{k^*:\infty}}^2)}{N^2\eta(1 - \alpha\eta\operatorname{tr}(\mathbf{H}))} \cdot (k^* + \frac{N^2\eta^2}{4}\sum_{i>k^*}\lambda_i^2)$$

### E.3 Specific Disrtibutions

**Proof of Corollary.** 1. if $\lambda_k = k^{-r}$, by the definition of $k^*$, we have $k^* = (\frac{N\eta}{2})^{1/r}$, then:

$$k^* + \frac{N^2\eta^2}{4}\sum_{k>k^*}\lambda_k^2 \simeq k^* + \frac{N\eta}{2}\sum_{k>k^*}\lambda_k \simeq (N\eta)^{\frac{1}{r}} + N\eta \cdot (N\eta)^{\frac{1-r}{r}} \simeq (N\eta)^{\frac{1}{r}}.$$

For the sake of simplicity, we assume $\|\mathbf{w}_0 - \mathbf{w}_*\|^2 \leq W^2$, then it hold that:

$$\text{Utility} \lesssim \frac{W^2}{N\eta} + (\frac{W^2}{N\eta} + \frac{W^2}{N\eta} + \frac{\sigma^2}{N} + \frac{\Gamma^2 f^2}{N^2}) \cdot (N\eta)^{\frac{1}{r}}.$$

If we choose the step size as follows:

$$\eta \simeq \min\{N^{-\frac{1}{1+r}}, N^{-\frac{1+r}{1+2r}}f^{-\frac{2r}{1+2r}}\},$$

which indicates:

$$\text{Utility} \lesssim (\sigma + W^2) \cdot N^{-\frac{r}{1+r}} + (W^2 + \Gamma^2) \cdot (f^{-2}N)^{-\frac{r}{1+2r}}.$$

Thus, it holds that $\text{Utility} = \widetilde{O}(N^{-\frac{r}{1+r}} \cdot (1 + (f^2 N^{\frac{r}{1+r}})^{\frac{r}{1+2r}}))$.

2. If $\lambda_k = e^{-k}$, we have $k^* = \log(N\eta)$, then:

$$k^* + \frac{N^2\eta^2}{4}\sum_{k>k^*}\lambda_k^2 \simeq k^* + N\eta\sum_{k>k^*}\lambda_k \simeq \log(N\eta) + \frac{N\eta}{2}\cdot(N\eta)^{-1} \simeq \log(N\eta).$$

Hence,

$$\text{Utility} \lesssim \frac{W^2}{N\eta} + (\frac{W^2}{N} + \frac{W^2}{N\eta} + \frac{\sigma^2}{N} + \frac{\Gamma^2 f^2}{N^2})\cdot\log(N\eta) \lesssim \frac{W^2}{N\eta} + \frac{W^2}{N} + \frac{\sigma^2}{N} + \frac{\Gamma^2 f^2}{N^2}.$$

If we choose the following step size $\eta \simeq \min\{1, \frac{1}{\sqrt{N}f}\}$, then we complete our results:

$$\text{Utility} \simeq \widetilde{O}(N^{-1}(1 + (\varepsilon^{-2}N)^{1/2})).$$

$\square$

